# Majorization for CRFs and Latent Likelihoods

**Tony Jebara**
Department of Computer Science
Columbia University
jebara@cs.columbia.edu

**Anna Choromanska**
Department of Electrical Engineering
Columbia University
aec2163@columbia.edu

## Abstract

The partition function plays a key role in probabilistic modeling including conditional random fields, graphical models, and maximum likelihood estimation. To optimize partition functions, this article introduces a quadratic variational upper bound. This inequality facilitates majorization methods: optimization of complicated functions through the iterative solution of simpler sub-problems. Such bounds remain efficient to compute even when the partition function involves a graphical model (with small tree-width) or in latent likelihood settings. For large-scale problems, low-rank versions of the bound are provided and outperform LBFGS as well as first-order methods. Several learning applications are shown and reduce to fast and convergent update rules. Experimental results show advantages over state-of-the-art optimization methods.

## 1   Introduction

The estimation of probability density functions over sets of random variables is a central problem in learning. Estimation often requires minimizing the partition function as is the case in conditional random fields (CRFs) and log-linear models [1, 2]. Training these models was traditionally done via iterative scaling and bound-majorization methods [3, 4, 5, 6, 1] which achieved monotonic convergence. These approaches were later surpassed by faster first-order methods [7, 8, 9] and then second-order methods such as LBFGS [10, 11, 12]. This article revisits majorization and repairs its slow convergence by proposing a tighter bound on the log-partition function. The improved majorization outperforms state-of-the-art optimization tools and admits multiple versatile extensions.

Many decomposition methods for conditional random fields and structured prediction have sought to render the learning and prediction problems more manageable [13, 14, 15]. Our decomposition, however, hinges on bounding and majorization: decomposing an optimization of complicated functions through the iterative solution of simpler sub-problems [16, 17]. A tighter bound provides convergent monotonic minimization while *outperforming* first- and second-order methods in practice[1]. The bound applies to graphical models [18], latent variable situations [17, 19, 20, 21] as well as high-dimensional settings [10]. It also accommodates convex constraints on the parameter space.

This article is organized as follows. Section 2 presents the bound and Section 3 uses it for majorization in CRFs. Extensions to latent likelihood are shown in Section 4. The bound is extended to graphical models in Section 5 and high dimensional problems in Section 6. Section 7 provides experiments and Section 8 concludes. The Supplement contains proofs and additional results.

## 2   Partition Function Bound

Consider a log-linear density model over discrete $y \in \Omega$

$$p(y|\boldsymbol{\theta}) \quad = \quad \frac{1}{Z(\boldsymbol{\theta})} h(y) \exp\left(\boldsymbol{\theta}^{\top} \mathbf{f}(y)\right)$$

which is parametrized by a vector $\boldsymbol{\theta} \in \mathbb{R}^d$ of dimensionality $d \in \mathbb{N}$. Here, $\mathbf{f} : \Omega \mapsto \mathbb{R}^d$ is any vector-valued function mapping an input $y$ to some arbitrary vector. The prior $h : \Omega \mapsto \mathbb{R}^+$ is a fixed non-negative measure. The partition function $Z(\boldsymbol{\theta})$ is a scalar that ensures that $p(y|\boldsymbol{\theta})$ normalizes, i.e. $Z(\boldsymbol{\theta}) = \sum_y h(y) \exp(\boldsymbol{\theta}^\top \mathbf{f}(y))$. Assume that the number of configurations of $y$ is $|\Omega| = n$ and is finite[2]. The partition function is clearly log-convex in $\boldsymbol{\theta}$ and a linear *lower-bound* is given via Jensen's inequality. This article contributes an analogous quadratic *upper-bound* on the partition function. Algorithm 1 computes[3] the bound's parameters and Theorem 1 shows the precise guarantee it provides.

---

**Algorithm 1** ComputeBound

---

Input Parameters $\boldsymbol{\theta}, \mathbf{f}(y), h(y) \ \forall y \in \Omega$

---

Init $z \to 0^+, \boldsymbol{\mu} = \mathbf{0}, \boldsymbol{\Sigma} = z\mathbf{I}$
For each $y \in \Omega$ {

$\quad \alpha \quad = h(y)\exp(\tilde{\boldsymbol{\theta}}^\top \mathbf{f}(y))$

$\quad \mathbf{l} \quad = \mathbf{f}(y) - \boldsymbol{\mu}$

$\quad \boldsymbol{\Sigma} += \frac{\tanh(\frac{1}{2}\log(\alpha/z))}{2\log(\alpha/z)}\mathbf{l}\mathbf{l}^\top$

$\quad \boldsymbol{\mu} += \frac{\alpha}{z+\alpha}\mathbf{l}$

$\quad z \ += \alpha \qquad$ }

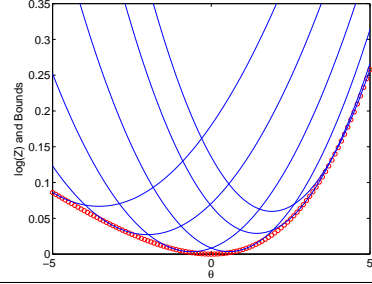

---

Output $z, \boldsymbol{\mu}, \boldsymbol{\Sigma}$

---

**Theorem 1** *Algorithm 1 finds* $z, \boldsymbol{\mu}, \boldsymbol{\Sigma}$ *such that* $z \exp(\frac{1}{2}(\boldsymbol{\theta} - \tilde{\boldsymbol{\theta}})^\top \boldsymbol{\Sigma}(\boldsymbol{\theta} - \tilde{\boldsymbol{\theta}}) + (\boldsymbol{\theta} - \tilde{\boldsymbol{\theta}})^\top \boldsymbol{\mu})$ *upper-bounds* $Z(\boldsymbol{\theta}) = \sum_y h(y)\exp(\boldsymbol{\theta}^\top \mathbf{f}(y))$ *for any* $\boldsymbol{\theta}, \tilde{\boldsymbol{\theta}}, \mathbf{f}(y) \in \mathbb{R}^d$ *and* $h(y) \in \mathbb{R}^+$ *for all* $y \in \Omega$.

**Proof 1 (Sketch, See Supplement for Formal Proof)** *Recall the bound* $\log(e^\theta + e^{-\theta}) \le c\theta^2$ *[22]. Obtain a multivariate variant* $\log(e^{\boldsymbol{\theta}^\top \mathbf{1}} + e^{-\boldsymbol{\theta}^\top \mathbf{1}})$. *Tilt the bound to handle* $\log(h_1 e^{\boldsymbol{\theta}^\top \mathbf{f}_1} + h_2 e^{\boldsymbol{\theta}^\top \mathbf{f}_2})$. *Add an additional exponential term to get* $\log(h_1 e^{\boldsymbol{\theta}^\top \mathbf{f}_1} + h_2 e^{\boldsymbol{\theta}^\top \mathbf{f}_2} + h_3 e^{\boldsymbol{\theta}^\top \mathbf{f}_3})$. *Iterate the last step to extend to* $n$ *elements in the summation.*

The bound improves previous inequalities and its proof is in the Supplement. It tightens [4, 19] since it avoids wasteful curvature tests (it uses duality theory to compare the bound and the optimized function rather than compare their Hessians). It generalizes [22] which only holds for $n = 2$ and $h(y)$ constant; it generalizes [23] which only handles a simplified one-dimensional case. The bound is computed using Algorithm 1 by iterating over the $y$ variables ("for each $y \in \Omega$") according to an arbitrary ordering via the bijective function $\pi : \Omega \mapsto \{1, \ldots, n\}$ which defines $i = \pi(y)$. The order in which we enumerate over $\Omega$ slightly varies the $\boldsymbol{\Sigma}$ in the bound (but not the $\boldsymbol{\mu}$ and $z$) when $|\Omega| > 2$. However, we empirically investigated the influence of various orderings on bound performance (in all the experiments presented in Section 7) and noticed no significant effect across ordering schemes. Recall that choosing $\boldsymbol{\Sigma} = \sum_y h(y)\exp(\tilde{\boldsymbol{\theta}}^\top \mathbf{f}(y))(\mathbf{f}(y) - \boldsymbol{\mu})(\mathbf{f}(y) - \boldsymbol{\mu})^\top$ with $\boldsymbol{\mu}$ and $z$ as in Algorithm 1 yields the second-order Taylor approximation (the Hessian) of the log-partition function. Algorithm 1 replaces a sum of log-linear models with a single log-quadratic model which makes monotonic majorization straightforward. The figure inside Algorithm 1 depicts the bound on $\log Z(\boldsymbol{\theta})$ for various choices of $\tilde{\boldsymbol{\theta}}$. If there are no constraints on the parameters (i.e. any $\boldsymbol{\theta} \in \mathbb{R}^d$ is admissible), a simple closed-form iterative update rule emerges: $\tilde{\boldsymbol{\theta}} \leftarrow \tilde{\boldsymbol{\theta}} - \boldsymbol{\Sigma}^{-1}\boldsymbol{\mu}$. Alternatively, if $\boldsymbol{\theta}$ must satisfy linear (convex) constraints it is straightforward to compute an update by solving a quadratic (convex) program. This update rule is interleaved with the bound computation.

## 3 Conditional Random Fields and Log-Linear Models

The partition function arises naturally in maximum entropy estimation or minimum relative entropy estimation (cf. Supplement) as well as in conditional extensions of the maximum entropy paradigm where the model is conditioned on an observed input. Such models are known as conditional random fields and have been useful for structured prediction problems [1, 24]. CRFs are given a data-set $\{(x_1, y_1), \ldots, (x_t, y_t)\}$ of independent identically-distributed (*iid*) input-output pairs where $y_j$ is

the observed sample in a (discrete) space $\Omega_j$ conditioned on the observed input $x_j$. A CRF defines a distribution over all $y \in \Omega_j$ (of which $y_j$ is a single element) as the log-linear model

$$p(y|x_j, \boldsymbol{\theta}) \;\; = \;\; \frac{1}{Z_{x_j}(\boldsymbol{\theta})} h_{x_j}(y) \exp(\boldsymbol{\theta}^\top \mathbf{f}_{x_j}(y))$$

where $Z_{x_j}(\boldsymbol{\theta}) \;=\; \sum_{y \in \Omega_j} h_{x_j}(y) \exp(\boldsymbol{\theta}^\top \mathbf{f}_{x_j}(y))$. For the $j$'th training pair, we are given a non-negative function $h_{x_j}(y) \in \mathbb{R}^+$ and a vector-valued function $\mathbf{f}_{x_j}(y) \in \mathbb{R}^d$ defined over the domain $y \in \Omega_j$. In this section, for simplicity, assume $n = \max_{j=1}^t |\Omega_{y_j}|$. Each partition function $Z_{x_j}(\boldsymbol{\theta})$ is a function of $\boldsymbol{\theta}$. The parameter $\boldsymbol{\theta}$ for CRFs is estimated by maximizing the regularized conditional log-likelihood[4] or log-posterior: $\sum_{j=1}^t \log p(y_j|x_j, \boldsymbol{\theta}) - \frac{t\lambda}{2} \|\boldsymbol{\theta}\|^2$ where $\lambda \in \mathbb{R}^+$ is a regularizer set using prior knowledge or cross-validation. Rewriting gives the objective of interest

$$J(\boldsymbol{\theta}) = \sum_{j=1}^t \log \frac{h_{x_j}(y_j)}{Z_{x_j}(\boldsymbol{\theta})} + \boldsymbol{\theta}^\top \mathbf{f}_{x_j}(y_j) - \frac{t\lambda}{2} \|\boldsymbol{\theta}\|^2. \tag{1}$$

If prior knowledge (or constraints) restrict the solution vector to a convex hull $\boldsymbol{\Lambda}$, the maximization problem becomes $\arg \max_{\boldsymbol{\theta} \in \boldsymbol{\Lambda}} J(\boldsymbol{\theta})$.

Algorithm 2 proposes a method for maximizing the regularized conditional likelihood $J(\boldsymbol{\theta})$ or, equivalently minimizing the partition function $Z(\boldsymbol{\theta})$. It solves the problem in Equation 1 subject to convex constraints by interleaving the quadratic bound with a quadratic programming procedure. Theorem 2 establishes the convergence of the algorithm and the proof is in the Supplement.

---

**Algorithm 2** ConstrainedMaximization

---

0: Input $x_j, y_j$ and functions $h_{x_j}, \mathbf{f}_{x_j}$ for $j = 1, \ldots, t$, regularizer $\lambda \in \mathbb{R}^+$ and convex hull $\boldsymbol{\Lambda} \subseteq \mathbb{R}^d$

1: Initialize $\boldsymbol{\theta}_0$ anywhere inside $\boldsymbol{\Lambda}$ and set $\tilde{\boldsymbol{\theta}} = \boldsymbol{\theta}_0$
   While not converged

2:  For $j = 1, \ldots, t$
       Get $\boldsymbol{\mu}_j, \boldsymbol{\Sigma}_j$ from $h_{x_j}, \mathbf{f}_{x_j}, \tilde{\boldsymbol{\theta}}$ via Algorithm 1

3:  Set $\tilde{\boldsymbol{\theta}} = \arg \min_{\boldsymbol{\theta} \in \boldsymbol{\Lambda}} \sum_j \frac{1}{2} (\boldsymbol{\theta} - \tilde{\boldsymbol{\theta}})^\top (\boldsymbol{\Sigma}_j + \lambda \mathbf{I})(\boldsymbol{\theta} - \tilde{\boldsymbol{\theta}}) + \sum_j \boldsymbol{\theta}^\top (\boldsymbol{\mu}_j - \mathbf{f}_{x_j}(y_j) + \lambda \tilde{\boldsymbol{\theta}})$

---

4: Output $\hat{\boldsymbol{\theta}} = \tilde{\boldsymbol{\theta}}$

---

**Theorem 2** *For any* $\boldsymbol{\theta}_0 \in \boldsymbol{\Lambda}$, *all* $\|\mathbf{f}_{x_j}(y)\| \leq r$ *and all* $|\Omega_j| \leq n$, *Algorithm 2 outputs a* $\hat{\boldsymbol{\theta}}$ *such that* $J(\hat{\boldsymbol{\theta}}) - J(\boldsymbol{\theta}_0) \geq (1 - \epsilon) \max_{\boldsymbol{\theta} \in \boldsymbol{\Lambda}} (J(\boldsymbol{\theta}) - J(\boldsymbol{\theta}_0))$ *in more than* $\left\lceil \log\left(\frac{1}{\epsilon}\right) / \log\left(1 + \frac{\lambda}{2r^2} (\sum_{i=1}^{n-1} \frac{\tanh(\log(i)/2)}{\log(i)})^{-1}\right) \right\rceil$ *iterations.*

The series $\sum_{i=1}^{n-1} \frac{\tanh(\log(i)/2)}{\log(i)} = \sum_{i=1}^{n-1} \frac{i-1}{(i+1)\log(i)}$ is the logarithmic integral which is $\mathrm{O}\left(\frac{n}{\log n}\right)$ asymptotically [26]. The next sections show how to handle hidden variables in the learning problem, exploit graphical modeling, and further accelerate the underlying algorithms.

## 4 Latent Conditional Likelihood

Section 3 showed how the partition function is useful for maximum conditional likelihood problems involving CRFs. In this section, maximum conditional likelihood is extended to the setting where some variables are latent. Latent models may provide more flexibility than fully observable models [21, 27, 28]. For instance, hidden conditional random fields were shown to outperform generative hidden-state and discriminative fully-observable models [21].

Consider the latent setting where we are given $t$ *iid* samples $x_1, \ldots, x_t$ from some unknown distribution $\bar{p}(x)$ and $t$ corresponding samples $y_1, \ldots, y_t$ drawn from identical conditional distributions $\bar{p}(y|x_1), \ldots, \bar{p}(y|x_t)$ respectively. Assume that the true generating distributions $\bar{p}(x)$ and $\bar{p}(y|x)$ are unknown. Therefore, we aim to estimate a conditional distribution $p(y|x)$ from some set of hypotheses that achieves high conditional likelihood given the data-set $\mathcal{D} = \{(x_1, y_1), \ldots, (x_t, y_t)\}$.

We will select this conditional distribution by assuming it emerges from a conditioned joint distribution over $x$ and $y$ as well as a hidden variable $m$ which is being marginalized as $p(y|x, \Theta) = \frac{\sum_m p(x,y,m|\Theta)}{\sum_{y,m} p(x,y,m|\Theta)}$. Here $m \in \Omega_m$ represents a discrete hidden variable, $x \in \Omega_x$ is an input and $y \in \Omega_y$ is a discrete output variable. The parameter $\Theta$ contains all parameters that explore the function class of such conditional distributions. The latent likelihood of the data $L(\Theta) = p(\mathcal{D}|\Theta)$ subsumes Equation 1 and is the new objective of interest

$$L(\Theta) = \prod_{j=1}^{t} p(y_j|x_j, \Theta) = \prod_{j=1}^{t} \frac{\sum_m p(x_j, y_j, m|\Theta)}{\sum_{y,m} p(x_j, y, m|\Theta)}. \tag{2}$$

A good choice of the parameters is one that achieves a large conditional likelihood value (or posterior) on the data set $\mathcal{D}$. Next, assume that each $p(x|y, m, \Theta)$ is an exponential family distribution

$$p(x|y, m, \Theta) = h(x) \exp\big(\boldsymbol{\theta}_{y,m}^\top \boldsymbol{\phi}(x) - a(\boldsymbol{\theta}_{y,m})\big)$$

where each conditional is specified by a function $h : \Omega_x \mapsto \mathbb{R}^+$ and a feature mapping $\boldsymbol{\phi} : \Omega_x \mapsto \mathbb{R}^d$ which are then used to derive the normalizer $a : \mathbb{R}^d \mapsto \mathbb{R}^+$. A parameter $\boldsymbol{\theta}_{y,m} \in \mathbb{R}^d$ selects a specific distribution. Multiply each exponential family term by an unknown marginal distribution called the *mixing proportions* $p(y, m|\pi) = \frac{\pi_{y,m}}{\sum_{y,m} \pi_{y,m}}$. This is parametrized by an unknown parameter $\pi = \{\pi_{y,m}\} \, \forall y, m$ where $\pi_{y,m} \in [0, \infty)$. Finally, the collection of all parameters is $\Theta = \{\boldsymbol{\theta}_{y,m}, \pi_{y,m}\} \, \forall y, m$. Thus, we have the complete likelihood $p(x, y, m|\Theta) = \frac{\pi_{y,m} h(x)}{\sum_{y,m} \pi_{y,m}} \exp\big(\boldsymbol{\theta}_{y,m}^\top \boldsymbol{\phi}(x) - a(\boldsymbol{\theta}_{y,m})\big)$. Insert this expression into Equation 2 and remove constant factors that appear in both denominator and numerator. Apply the change of variables $\exp(\nu_{y,m}) = \pi_{y,m} \exp(-a(\boldsymbol{\theta}_{y,m}))$ and rewrite the objective as a function[5] of a vector $\boldsymbol{\theta}$:

$$L(\Theta) = \prod_{j=1}^{t} \frac{\sum_m \exp\big(\boldsymbol{\theta}_{y_j,m}^\top \boldsymbol{\phi}(x_j) + \nu_{y_j,m}\big)}{\sum_{y,m} \exp\big(\boldsymbol{\theta}_{y,m}^\top \boldsymbol{\phi}(x_j) + \nu_{y,m}\big)} = \prod_{j=1}^{t} \frac{\sum_m \exp\big(\boldsymbol{\theta}^\top \mathbf{f}_{j,y_j,m}\big)}{\sum_{y,m} \exp\big(\boldsymbol{\theta}^\top \mathbf{f}_{j,y,m}\big)}.$$

The last equality emerges by rearranging all $\Theta$ parameters as a vector $\boldsymbol{\theta} \in \mathbb{R}^{|\Omega_y||\Omega_m|(d+1)}$ equal to $[\boldsymbol{\theta}_{1,1}^\top \, \nu_{1,1} \, \boldsymbol{\theta}_{1,2}^\top \, \nu_{1,2} \, \cdots \, \boldsymbol{\theta}_{|\Omega_y|,|\Omega_m|}^\top \, \nu_{|\Omega_y|,|\Omega_m|}]^\top$ and introducing $\mathbf{f}_{j,\hat{y},\hat{m}} \in \mathbb{R}^{|\Omega_y||\Omega_m|(d+1)}$ defined as $\big[[\boldsymbol{\phi}(x_j)^\top 1]\delta[(\hat{y},\hat{m})=(1,1)] \, \cdots \, [\boldsymbol{\phi}(x_j)^\top 1]\delta[(\hat{y},\hat{m})=(|\Omega_y|,|\Omega_m|)]\big]^\top$ (thus the feature vector $[\boldsymbol{\phi}(x_j)^\top 1]^\top$ is positioned appropriately in the longer $\mathbf{f}_{j,\hat{y},\hat{m}}$ vector which is elsewhere zero). We will now find a variational lower bound on $L(\boldsymbol{\theta}) \geq Q(\boldsymbol{\theta}, \tilde{\boldsymbol{\theta}})$ which is tight when $\boldsymbol{\theta} = \tilde{\boldsymbol{\theta}}$ such that $L(\tilde{\boldsymbol{\theta}}) = Q(\tilde{\boldsymbol{\theta}}, \tilde{\boldsymbol{\theta}})$. We proceed by bounding each numerator and each denominator in the product over $j = 1, \ldots, t$. Apply Jensen's inequality to lower bound each numerator term as

$$\sum_m \exp\big(\boldsymbol{\theta}^\top \mathbf{f}_{j,y_j,m}\big) \geq e^{\boldsymbol{\theta}^\top \sum_m \eta_{j,m} \mathbf{f}_{j,y_j,m} - \sum_m \eta_{j,m} \log \eta_{j,m}}$$

where $\eta_{j,m} = (e^{\tilde{\boldsymbol{\theta}}^\top \mathbf{f}_{j,y_j,m}})/(\sum_{m'} e^{\tilde{\boldsymbol{\theta}}^\top \mathbf{f}_{j,y_j,m'}})$. Algorithm 1 then bounds the denominator

$$\sum_{y,m} \exp\big(\boldsymbol{\theta}^\top \mathbf{f}_{j,y,m}\big) \leq z_j e^{\frac{1}{2}(\boldsymbol{\theta}-\tilde{\boldsymbol{\theta}})^\top \boldsymbol{\Sigma}_j (\boldsymbol{\theta}-\tilde{\boldsymbol{\theta}}) + (\boldsymbol{\theta}-\tilde{\boldsymbol{\theta}})^\top \boldsymbol{\mu}_j}.$$

The overall lower bound on the likelihood is then

$$Q(\boldsymbol{\theta}, \tilde{\boldsymbol{\theta}}) = L(\tilde{\boldsymbol{\theta}}) e^{-\frac{1}{2}(\boldsymbol{\theta}-\tilde{\boldsymbol{\theta}})^\top \tilde{\boldsymbol{\Sigma}} (\boldsymbol{\theta}-\tilde{\boldsymbol{\theta}}) - (\boldsymbol{\theta}-\tilde{\boldsymbol{\theta}})^\top \tilde{\boldsymbol{\mu}}}$$

where $\tilde{\boldsymbol{\Sigma}} = \sum_{j=1}^{t} \boldsymbol{\Sigma}_j$ and $\tilde{\boldsymbol{\mu}} = \sum_{j=1}^{t} (\boldsymbol{\mu}_j - \sum_m \eta_{j,m} \mathbf{f}_{j,y_j,m})$. The right hand side is simply an exponentiated quadratic function in $\boldsymbol{\theta}$ which is easy to maximize. This yields an iterative scheme similar to Algorithm 2 for monotonically maximizing latent conditional likelihood.

## 5  Graphical Models for Large $n$

The bounds in the previous sections are straightforward to compute when $\Omega$ is small. However, for graphical models, enumerating over $\Omega$ can be daunting. This section provides faster algorithms

that recover the bound efficiently for graphical models of bounded tree-width. A graphical model represents the factorization of a probability density function. This article will consider the factor graph notation of a graphical model. A factor graph is a bipartite graph $\mathcal{G} = (V, W, E)$ with variable vertices $V = \{1, \ldots, k\}$, factor vertices $W = \{1, \ldots, m\}$ and a set of edges $E$ between $V$ and $W$. In addition, define a set of random variables $Y = \{y_1, \ldots, y_k\}$ each associated with the elements of $V$ and a set of non-negative scalar functions $\Psi = \{\psi_1, \ldots, \psi_m\}$ each associated with the elements of $W$. The factor graph implies that $p(Y)$ factorizes as $p(y_1, \ldots, y_k) = \frac{1}{Z} \prod_{c \in W} \psi_c(Y_c)$ where $Z$ is a normalizing partition function (the dependence on parameters is suppressed here) and $Y_c$ is a subset of the random variables that are associated with the neighbors of node $c$. In other words, $Y_c = \{y_i | i \in \mathsf{Ne}(c)\}$ where $\mathsf{Ne}(c)$ is the set of vertices that are neighbors of $c$. Inference in graphical models requires the evaluation and the optimization of $Z$. These computations can be NP-hard in general yet are efficient when $\mathcal{G}$ satisfies certain properties (low tree-width). Consider a log-linear model (a function class) indexed by a parameter $\boldsymbol{\theta} \in \boldsymbol{\Lambda}$ in a convex hull $\boldsymbol{\Lambda} \subseteq \mathbb{R}^d$ as follows

$$ p(Y|\boldsymbol{\theta}) \;=\; \frac{1}{Z(\boldsymbol{\theta})} \prod_{c \in W} h_c(Y_c) \exp\left(\boldsymbol{\theta}^\top \mathbf{f}_c(Y_c)\right) $$

where $Z(\boldsymbol{\theta}) = \sum_Y \prod_{c \in W} h_c(Y_c) \exp\left(\boldsymbol{\theta}^\top \mathbf{f}_c(Y_c)\right)$. The model is defined by a set of vector-valued functions $\mathbf{f}_c(Y_c) \in \mathbb{R}^d$ and scalar-valued functions $h_c(Y_c) \in \mathbb{R}^+$. Choosing a function from the function class hinges on estimating $\boldsymbol{\theta}$ by optimizing $Z(\boldsymbol{\theta})$. However, Algorithm 1 may be inapplicable due to the large number of configurations in $Y$. Instead, consider a more efficient surrogate algorithm which computes the same bound parameters by efficiently exploiting the factorization of the graphical model. This is possible since exponentiated quadratics are closed under multiplication and the required bound computations distribute nicely across decomposable graphical models.

---

**Algorithm 3** JunctionTreeBound

---

Input     Reverse-topological tree $\mathcal{T}$ with $c = 1, \ldots, m$ factors $h_c(Y_c) \exp(\tilde{\boldsymbol{\theta}}^\top \mathbf{f}_c(Y_c))$ and $\tilde{\boldsymbol{\theta}} \in \mathbb{R}^d$

---

For $c = 1, \ldots, m$

     If $(c < m)$ $\{Y_{both} = Y_c \cap Y_{pa(c)},\ Y_{solo} = Y_c \setminus Y_{pa(c)}\}$

     Else $\{Y_{both} = \{\},\ Y_{solo} = Y_c\}$

     For each $u \in Y_{both}$

       $\{$ Initialize $z_{c|x} \leftarrow 0^+,\ \boldsymbol{\mu}_{c|x} = \mathbf{0},\ \boldsymbol{\Sigma}_{c|x} = z_{c|x}\mathbf{I}$

       For each $v \in Y_{solo}$

       $\{$    $w = u \otimes v;$

         $\alpha_w = h_c(w) e^{\tilde{\boldsymbol{\theta}}^\top \mathbf{f}_c(w)} \prod_{b \in ch(c)} z_{b|w};$    $\mathbf{l}_w = \mathbf{f}_c(w) - \boldsymbol{\mu}_{c|u} + \sum_{b \in ch(c)} \boldsymbol{\mu}_{b|w};$

         $\boldsymbol{\Sigma}_{c|u} = \sum_{b \in ch(c)} \boldsymbol{\Sigma}_{b|w} + \frac{\tanh(\frac{1}{2}\log(\frac{\alpha_w}{z_{c|u}}))}{2\log(\frac{\alpha_w}{z_{c|u}})} \mathbf{l}_w \mathbf{l}_w^\top;$    $\boldsymbol{\mu}_{c|u} = \frac{\alpha_w}{z_{c|u} + \alpha_w} \mathbf{l}_w;$    $z_{c|u} = \alpha_w;$ $\}\}$

---

Output     Bound as $z = z_m,\ \boldsymbol{\mu} = \boldsymbol{\mu}_m,\ \boldsymbol{\Sigma} = \boldsymbol{\Sigma}_m$

---

Begin by assuming that the graphical model in question is a *junction tree* and satisfies the running intersection property [18]. In Algorithm 3 (the Supplement provides a proof of its correctness), take $ch(c)$ to be the set of children-cliques of clique $c$ and $pa(c)$ to be the parent of $c$. Note that the algorithm enumerates over $u \in Y_{pa(c)} \cap Y_c$ and $v \in Y_c \setminus Y_{pa(c)}$. The algorithm stores a quadratic bound for each configuration of $u$ (where $u$ is the set of variables in common across both clique $c$ and its parent). It then forms the bound by summing over $v \in Y_c \setminus Y_{pa(c)}$, each configuration of each variable a clique $c$ has that is *not* shared with its parent clique. The algorithm also collects precomputed bounds from children of $c$. Also define $w = u \otimes v \in Y_c$ as the conjunction of both indexing variables $u$ and $v$. Thus, the two inner for loops enumerate over all configurations $w \in Y_c$ of each clique. Note that $w$ is used to query the children $b \in ch(c)$ of a clique $c$ to report their bound parameters $z_{b|w}, \boldsymbol{\mu}_{b|w}, \boldsymbol{\Sigma}_{b|w}$. This is done for each configuration $w$ of the clique $c$. Note, however, that not every variable in clique $c$ is present in each child $b$ so only the variables in $w$ that intersect $Y_b$ are relevant in indexing the parameters $z_{b|w}, \boldsymbol{\mu}_{b|w}, \boldsymbol{\Sigma}_{b|w}$ and the remaining variables do not change the values of $z_{b|w}, \boldsymbol{\mu}_{b|w}, \boldsymbol{\Sigma}_{b|w}$.

Algorithm 3 is efficient in the sense that computations involve enumerating over all configurations of each clique in the junction tree rather than over all configurations of $Y$. This shows that the

computation involved is $O(\sum_c |Y_c|)$ rather than $O(|\Omega|)$ as in Algorithm 1. Thus, for estimating the computational efficiency of various algorithms in this article, take $n = \sum_c |Y_c|$ for the graphical model case rather than $n = |\Omega|$. Algorithm 3 is a simple extension of the known recursions that are used to compute the partition function and its gradient vector. Thus, in addition to the $\boldsymbol{\Sigma}$ matrix which represents the curvature of the bound, Algorithm 3 is recovering the partition function value $z$ and the gradient since $\boldsymbol{\mu} = \left. \frac{\partial \log Z(\boldsymbol{\theta})}{\partial \boldsymbol{\theta}} \right|_{\boldsymbol{\theta} = \tilde{\boldsymbol{\theta}}}$.

## 6   Low-Rank Bounds for Large $d$

In many realistic situations, the dimensionality $d$ is large and this prevents the storage and inversion of the matrix $\boldsymbol{\Sigma}$. We next present a low-rank extension that can be applied to any of the algorithms presented so far. As an example, consider Algorithm 4 which is a low-rank incarnation of Algorithm 2. Each iteration of Algorithm 2 requires $O(tnd^2 + d^3)$ time since step 2 computes several $\boldsymbol{\Sigma}_j \in \mathbb{R}^{d \times d}$ matrices and 3 performs inversion. Instead, the new algorithm provides a low-rank version of the bound which still majorizes the log-partition function but requires only $\tilde{O}(tnd)$ complexity (putting it on par with LBFGS). First, note that step 3 in Algo-

---

**Algorithm 4** LowRankBound

---

Input Parameter $\tilde{\boldsymbol{\theta}}$, regularizer $\lambda \in \mathbb{R}^+$, model $\mathbf{f}_t(y) \in \mathbb{R}^d$ and $h_t(y) \in \mathbb{R}^+$ and rank $k \in \mathbb{N}$

---

Initialize $\mathbf{S} = \mathbf{0} \in \mathbb{R}^{k \times k}$, $\mathbf{V} = $ orthonormal $\in \mathbb{R}^{k \times d}$, $\mathbf{D} = t\lambda\mathbf{I} \in \mathsf{diag}(\mathbb{R}^{d \times d})$
For each $t$ { Set $z \to 0^+$; $\boldsymbol{\mu} = \mathbf{0}$;
$\quad$ For each y{
$$\alpha = h_t(y)e^{\tilde{\boldsymbol{\theta}}^\top \mathbf{f}_t(y)}; \quad \mathbf{r} = \frac{\sqrt{\tanh(\frac{1}{2}\log(\frac{\alpha}{z}))}}{\sqrt{2\log(\frac{\alpha}{z})}}(\mathbf{f}_t(y) - \boldsymbol{\mu});$$
$\qquad$ For $i = 1, \ldots, k$ : $\mathbf{p}(i) = \mathbf{r}^\top \mathbf{V}(i, \cdot)$; $\mathbf{r} = \mathbf{r} - \mathbf{p}(i)\mathbf{V}(i, \cdot)$;
$\qquad$ For $i = 1, \ldots, k$ : For $j = 1, \ldots, k$ : $\mathbf{S}(i, j) = \mathbf{S}(i, j) + \mathbf{p}(i)\mathbf{p}(j)$;
$\qquad$ $\mathbf{Q}^\top \mathbf{A} \mathbf{Q} = \mathrm{svd}(\mathbf{S})$; $\mathbf{S} \leftarrow \mathbf{A}$; $\mathbf{V} \leftarrow \mathbf{Q}\mathbf{V}$;
$\qquad$ $\mathbf{s} = [\mathbf{S}(1,1), \ldots, \mathbf{S}(k,k), \|\mathbf{r}\|^2]^\top$; $\tilde{k} = \arg\min_{i=1,\ldots,k+1} \mathbf{s}(i)$;
$\qquad$ if $(\tilde{k} \leq k)$ { $\quad \mathbf{D} = \mathbf{D} + \mathbf{S}(\tilde{k}, \tilde{k})\mathbf{1}^\top |\mathbf{V}(j, \cdot)|\,\mathsf{diag}(|\mathbf{V}(k, \cdot)|)$;
$\qquad\qquad\qquad$ $\mathbf{S}(\tilde{k}, \tilde{k}) = \|\mathbf{r}\|^2$; $\mathbf{r} = \|\mathbf{r}\|^{-1}\mathbf{r}$; $\mathbf{V}(k, \cdot) = \mathbf{r}$; }
$\qquad$ else $\qquad$ { $\quad \mathbf{D} = \mathbf{D} + \mathbf{1}^\top |\mathbf{r}|\mathsf{diag}(|\mathbf{r}|)$; } }
$\qquad$ $\boldsymbol{\mu} += \frac{\alpha}{z + \alpha}(\mathbf{f}_t(y) - \boldsymbol{\mu})$; $z += \alpha$; $\qquad\qquad\qquad$ } }

---

Output $\mathbf{S} \in \mathsf{diag}(\mathbb{R}^{k \times k})$, $\mathbf{V} \in \mathbb{R}^{k \times d}$, $\mathbf{D} \in \mathsf{diag}(\mathbb{R}^{d \times d})$

---

rithm 2 can be written as $\tilde{\boldsymbol{\theta}} = \tilde{\boldsymbol{\theta}} - \boldsymbol{\Sigma}^{-1}\mathbf{u}$ where $\mathbf{u} = t\lambda\tilde{\boldsymbol{\theta}} + \sum_{j=1}^t \boldsymbol{\mu}_j - \mathbf{f}_{x_j}(y_j)$. Clearly, Algorithm 1 can recover $\mathbf{u}$ by *only* computing $\boldsymbol{\mu}_j$ for $j = 1, \ldots, t$ and skipping all steps involving matrices. This merely requires $O(tnd)$ work. Second, we store $\boldsymbol{\Sigma}$ using a low-rank representation $\mathbf{V}^\top \mathbf{S} \mathbf{V} + \mathbf{D}$ where $\mathbf{V} \in \mathbb{R}^{k \times d}$ is orthonormal, $\mathbf{S} \in \mathbb{R}^{k \times k}$ is positive semi-definite, and $\mathbf{D} \in \mathbb{R}^{d \times d}$ is non-negative diagonal. Rather than increment the matrix by a rank one update of the form $\boldsymbol{\Sigma}_i = \boldsymbol{\Sigma}_{i-1} + \mathbf{r}_i \mathbf{r}_i^\top$ where $\mathbf{r}_i = \sqrt{\frac{\tanh(\frac{1}{2}\log(\alpha/z))}{2\log(\alpha/z)}}(\mathbf{f}_i - \boldsymbol{\mu}_i)$, simply project $\mathbf{r}_i$ onto each eigenvector in $\mathbf{V}$ and update matrix $\mathbf{S}$ and $\mathbf{V}$ via a singular value decomposition ($O(k^3)$ work). After removing $k$ such projections, the remaining residual from $\mathbf{r}_i$ forms a new eigenvector $\mathbf{e}_{k+1}$ and its magnitude forms a new singular value. The resulting rank $(k+1)$ system is orthonormal with $(k+1)$ singular values. We discard its smallest singular value and corresponding eigenvector to revert back to an order $k$ eigensystem. However, instead of merely *discarding* we can *absorb* the smallest singular value and eigenvector into the $\mathbf{D}$ component by bounding the remaining outer-product with a diagonal term. This provides a guaranteed overall upper bound in $\tilde{O}(tnd)$ ($k$ is assumed to be logarithmic with dimension $d$). Finally, to invert $\boldsymbol{\Sigma}$, we apply the Woodbury formula: $\boldsymbol{\Sigma}^{-1} = \mathbf{D}^{-1} + \mathbf{D}^{-1}\mathbf{V}^\top(\mathbf{S}^{-1} + \mathbf{V}\mathbf{D}^{-1}\mathbf{V}^\top)^{-1}\mathbf{V}\mathbf{D}^{-1}$ which only requires $O(k^3)$ work. A proof of correctness for Algorithm 4 can be found in the Supplement.

## 7   Experiments

We first focus on the logistic regression task and compare the performance of the bound (using the low-rank Algorithm 2) with first-order and second order methods such as LBFGS, conjugate gradient (CG) and steepest descent (SD). We use 4 benchmark data-sets: the SRBCT and Tumors

data-sets from [29] as well as the Text and SecStr data-sets from http://olivier.chapelle.cc/ssl-book/benchmarks.html. For all experiments in this section, the setup is as follows. Each data-set is split into training (90%) and testing (10%) parts. All implementations are run on the same hardware with C++ code. The termination criterion for all algorithms is a change in estimated parameter or function values smaller than $10^{-6}$ (with a ceiling on the number of iterations of $10^6$). Results are averaged over 10 random initializations close to **0**. The regularization parameter $\lambda$, when used, was chosen through crossvalidation. In Table 1 we report times in seconds and the number of iterations for each algorithm (including LBFGS) to achieve the LBFGS termination solution modulo a small constant $\epsilon$ (set to $10^{-4}$). Table 1 also provides data-set sizes and regularization values. The first 4 columns in Table 1 provide results for this experiment.

| Data-set | SRBCT | | Tumors | | Text | | SecStr | | CoNLL | | PennTree | |
|---|---|---|---|---|---|---|---|---|---|---|---|---|
| Size | $n = 4$ | | $n = 26$ | | $n = 2$ | | $n = 2$ | | $m = 9$ | | $m = 45$ | |
| | $t = 83$ | | $t = 308$ | | $t = 1500$ | | $t = 83679$ | | $t = 1000$ | | $t = 1000$ | |
| | $d = 9236$ | | $d = 390260$ | | $d = 23922$ | | $d = 632$ | | $d = 33615$ | | $d = 14175$ | |
| | $\lambda = 10^1$ | | $\lambda = 10^1$ | | $\lambda = 10^2$ | | $\lambda = 10^1$ | | $\lambda = 10^1$ | | $\lambda = 10^1$ | |
| Algorithm | time | iter | time | iter | time | iter | time | iter | time | iter | time | iter |
| LBFGS | 6.10 | 42 | 3246.83 | 8 | 15.54 | 7 | 881.31 | 47 | 25661.54 | 17 | 62848.08 | 7 |
| SD | 7.27 | 43 | 18749.15 | 53 | 153.10 | 69 | 1490.51 | 79 | 93821.72 | 12 | 156319.31 | 12 |
| CG | 40.61 | 100 | 14840.66 | 42 | 57.30 | 23 | 667.67 | 36 | 88973.93 | 23 | 76332.39 | 18 |
| Bound | **3.67** | **8** | **1639.93** | **4** | **6.18** | **3** | **27.97** | **9** | **16445.93** | **4** | **27073.42** | **2** |

Table 1: Time in seconds and iterations required to obtain within $\epsilon$ of the LBFGS solution (where $\epsilon = 10^{-4}$) for logistic regression problems (on SRBCT, Tumors, Text and SecStr data-sets where $n$ is the number of classes) and Markov CRF problems (on CoNLL and PennTree data-sets, where $m$ is the number of classes). Here, $t$ is the total number of samples (training and testing), $d$ is the dimensionality of the feature vector and $\lambda$ is the cross-validated regularization setting.

Structured prediction problems are explored using two popular data-sets. The first one contains Spanish news wire articles from the a session of the CoNLL 2002 conference. This corpus involves a named entity recognition problem and consists of sentences where each word is annotated with one of $m = 9$ possible labels. The second task is from the PennTree Bank. This corpus involves a tagging problem and consists of sentences where each word is labeled with one of $m = 45$ possible parts-of-speech. A conditional random field is estimated with a Markov chain structure to give word labels a sequential dependence. The features describing the words are constructed as in [30]. Two last columns of Table 1 provide results for this experiment. We used the low-rank version of Algorithm 3. In both experiments, the bound always remained fastest as indicated in bold.

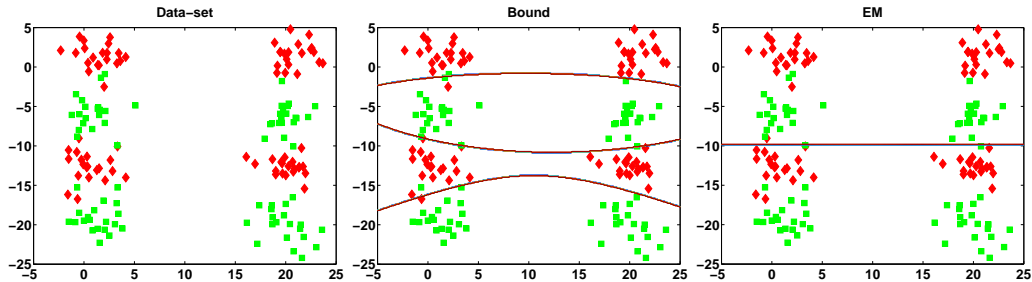

Figure 1: Classification boundaries using the bound and EM for a toy latent likelihood problem.

We next performed experiments with maximum latent conditional likelihood problems. We denote by $m$ the number of hidden variables. Due to the non-concavity of this objective, we are most interested in finding good local maxima. We start with a simple toy experiment from [19] comparing the bound to the expectation-maximization (EM) algorithm in the binary classification problem presented on the left image of Figure 1. The model incorrectly uses only 2 Gaussians per class while the data is generated using 8 Gaussians total. On Figure 1 we show the decision boundary obtained using the bound (with $m = 2$) and EM. EM performs as well as random chance guessing while the bound classifies the data very well. The average test log-likelihood obtained by EM was -1.5e+06 while the bound obtained -21.8.

We next compared the algorithms (the bound, Newton-Raphson, BFGS, CG and SD) in maximum latent conditional likelihood problems on five benchmark data-sets. These included four UCI data-sets[6] (ion, bupa, hepatitis and wine) and the previously used SRBCT data-set. The feature mapping used was $\phi(x) = \mathbf{x} \in \mathbb{R}^d$ which corresponds to a mixture *Gaussian-gated* logistic regressions (obtained by conditioning a mixture of $m$ Gaussians per class). We used a value of $\lambda = 0$ throughout the latent experiments. We explored setting $m \in \{1, 2, 3, 4\}$. Table 2 shows the testing latent log-likelihood at convergence for $m$ chosen through cross-validation (the Supplement contains a more complete table). In bold, we show the algorithm that obtained the highest testing log-likelihood. The bound is the best performer overall and finds better solutions in less time. Figure 2 depicts the convergence on ion, hepatitis and SRBCT data sets.

| Data-set | ion | bupa | hepatitis | wine | SRBCT |
|---|---|---|---|---|---|
| Algorithm | m = 3 | m = 2 | m = 2 | m = 3 | m = 4 |
| BFGS | -5.88 | -21.78 | -5.28 | -1.79 | -6.06 |
| SD | -5.56 | -21.74 | -5.14 | -1.37 | -5.61 |
| CG | -5.57 | -21.81 | -4.84 | -0.95 | -5.76 |
| Newton | -5.95 | -21.85 | -5.50 | -0.71 | -5.54 |
| Bound | **-4.18** | **-19.95** | **-4.40** | **-0.48** | **-0.11** |

Table 2: Test log-likelihood at convergence for ion, bupa, hepatitis, wine and SRBCT data-sets.

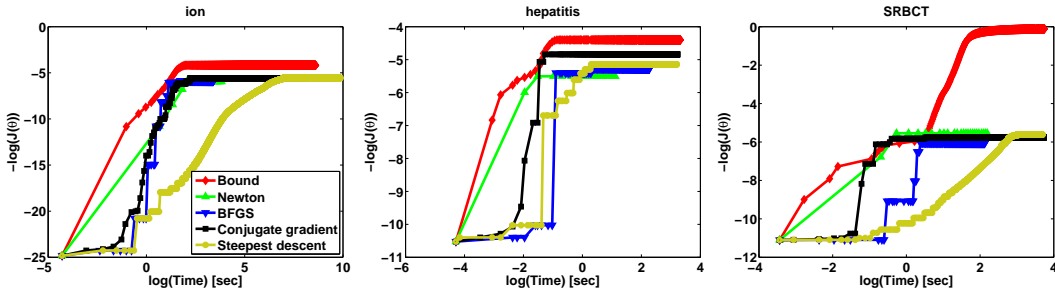

Figure 2: Convergence of test latent log-likelihood on ion, hepatitis and SRBCT data-sets.

# 8   Discussion

A simple quadratic upper bound for the partition function of log-linear models was proposed and makes majorization approaches competitive with state-of-the-art first- and second-order optimization methods. The bound is efficiently recoverable for graphical models and admits low-rank variants for high-dimensional data. It allows faster and monotonically convergent majorization in CRF learning and maximum latent conditional likelihood problems (where it also finds better local maxima). Future work will explore *intractable* partition functions where likelihood evaluation is hard but bound maximization may remain feasible. Furthermore, the majorization approach will be applied in stochastic [31] and distributed optimization settings.

## Acknowledgments

The authors thank A. Smola, M. Collins, D. Kanevsky and the referees for valuable feedback.

## Footnotes

[1]Recall that some second-order methods like Newton-Raphson are *not* monotonic and may even fail to converge for convex cost functions [4] unless, of course, line searches are used.

[2] Here, assume $n$ is enumerable. Later, for larger spaces use $\mathrm{O}(n)$ to denote the time to compute $Z$.

[3] By continuity, take $\tanh(\frac{1}{2}\log(1))/(2\log(1)) = \frac{1}{4}$ and $\lim_{z\to 0^+} \tanh(\frac{1}{2}\log(\alpha/z))/(2\log(\alpha/z)) = 0$.

[4]Alternatively, variational Bayesian approaches can be used instead of maximum likelihood via expectation propagation (EP) or power EP [25]. These, however, assume Gaussian posterior distributions over parameters, require approximations, are computationally expensive and are not necessarily more efficient than BFGS.

[5]It is now easy to regularize $L(\boldsymbol{\theta})$ by adding $-\frac{t\lambda}{2}\|\boldsymbol{\theta}\|^2$.

## References

[1] J. Lafferty, A. McCallum, and F. Pereira. Conditional random fields: Probabilistic models for segmenting and labeling sequence data. In *ICML*, 2001.

[2] A. Globerson, T. Koo, X. Carreras, and M. Collins. Exponentiated gradient algorithms for log-linear structured prediction. In *ICML*, 2007.

[3] J. Darroch and D. Ratcliff. Generalized iterative scaling for log-linear models. *Annals of Math. Stat.*, 43:1470–1480, 1972.

---

[6]Downloaded from http://archive.ics.uci.edu/ml/

[4] D. Bohning and B. Lindsay. Monotonicity of quadratic approximation algorithms. *Ann. Inst. Statist. Math.*, 40:641–663, 1988.

[5] A. Berger. The improved iterative scaling algorithm: A gentle introduction. Technical report, 1997.

[6] S. Della Pietra, V. Della Pietra, and J. Lafferty. Inducing features of random fields. *IEEE PAMI*, 19(4), 1997.

[7] R. Malouf. A comparison of algorithms for maximum entropy parameter estimation. In *CoNLL*, 2002.

[8] H. Wallach. Efficient training of conditional random fields. Master's thesis, University of Edinburgh, 2002.

[9] F. Sha and F. Pereira. Shallow parsing with conditional random fields. In *NAACL*, 2003.

[10] C. Zhu, R. Byrd, P Lu, and J. Nocedal. Algorithm 778: L-BFGS-B: Fortran subroutines for large-scale bound-constrained optimization. *TOMS*, 23(4), 1997.

[11] S. Benson and J. More. A limited memory variable metric method for bound constrained optimization. Technical report, Argonne National Laboratory, 2001.

[12] G. Andrew and J. Gao. Scalable training of $\ell 1$-regularized log-linear models. In *ICML*, 2007.

[13] D. Roth. Integer linear programming inference for conditional random fields. In *ICML*, 2005.

[14] Y. Mao and G. Lebanon. Generalized isotonic conditional random fields. *Machine Learning*, 77:225–248, 2009.

[15] C. Sutton and A. McCallum. Piecewise training for structured prediction. *Machine Learning*, 77:165–194, 2009.

[16] J. De Leeuw and W. Heiser. *Convergence of correction matrix algorithms for multidimensional scaling*, chapter Geometric representations of relational data. 1977.

[17] A. Dempster, N. Laird, and D. Rubin. Maximum likelihood from incomplete data via the EM algorithm. *J. of the Royal Stat. Soc.*, B-39, 1977.

[18] M. Wainwright and M Jordan. Graphical models, exponential families and variational inference. *Foundations and Trends in Machine Learning*, 1(1-2):1–305, 2008.

[19] T. Jebara and A. Pentland. On reversing Jensen's inequality. In *NIPS*, 2000.

[20] J. Salojarvi, K Puolamaki, and S. Kaski. Expectation maximization algorithms for conditional likelihoods. In *ICML*, 2005.

[21] A. Quattoni, S. Wang, L. P. Morency, M. Collins, and T. Darrell. Hidden conditional random fields. *IEEE PAMI*, 29(10):1848–1852, October 2007.

[22] T. Jaakkola and M. Jordan. Bayesian parameter estimation via variational methods. *Statistics and Computing*, 10:25–37, 2000.

[23] G. Bouchard. Efficient bounds for the softmax and applications to approximate inference in hybrid models. In *NIPS AIHM Workshop*, 2007.

[24] B. Taskar, C. Guestrin, and D. Koller. Max margin Markov networks. In *NIPS*, 2004.

[25] Y. Qi, M. Szummer, and T. P. Minka. Bayesian conditional random fields. In *AISTATS*, 2005.

[26] T. Bromwich and T. MacRobert. *An Introduction to the Theory of Infinite Series*. Chelsea, 1991.

[27] S. B. Wang, A. Quattoni, L.-P. Morency, and D. Demirdjian. Hidden conditional random fields for gesture recognition. In *CVPR*, 2006.

[28] Y. Wang and G. Mori. Max-margin hidden conditional random fields for human action recognition. In *CVPR*, pages 872–879. IEEE, 2009.

[29] F. Bach, R. Jenatton, J. Mairal, and G. Obozinski. *Optimization for Machine Learning*, chapter Convex optimization with sparsity-inducing norms. MIT Press, 2011.

[30] Y. Altun, I. Tsochantaridis, and T. Hofmann. Hidden Markov support vector machines. In *ICML*, 2003.

[31] SVN. Vishwanathan, N. Schraudolph, M. Schmidt, and K. Murphy. Accelerated training of conditional random fields with stochastic gradient methods. In *ICML*, 2006.

[32] T. Jebara. Multitask sparsity via maximum entropy discrimination. *JMLR*, 12:75–110, 2011.

# Majorization for CRFs and Latent Likelihoods
## (Supplementary Material)

**Tony Jebara**
Department of Computer Science
Columbia University
jebara@cs.columbia.edu

**Anna Choromanska**
Department of Electrical Engineering
Columbia University
aec2163@columbia.edu

## Abstract

This supplement presents additional details in support of the full article. These include the application of the majorization method to maximum entropy problems. It also contains proofs of the various theorems, in particular, a guarantee that the bound majorizes the partition function. In addition, a proof is provided guaranteeing convergence on (non-latent) maximum conditional likelihood problems. The supplement also contains supporting lemmas that show the bound remains applicable in constrained optimization problems. The supplement then proves the soundness of the junction tree implementation of the bound for graphical models with large $n$. It also proves the soundness of the low-rank implementation of the bound for problems with large $d$. Finally, the supplement contains additional experiments and figures to provide further empirical support for the majorization methodology.

## Supplement for Section 2

**Proof of Theorem 1** Rewrite the partition function as a sum over the integer index $j = 1, \ldots, n$ under the random ordering $\pi : \Omega \mapsto \{1, \ldots, n\}$. This defines $j = \pi(y)$ and associates $h$ and $\mathbf{f}$ with $h_j = h(\pi^{-1}(j))$ and $\mathbf{f}_j = \mathbf{f}(\pi^{-1}(j))$. Next, write $Z(\boldsymbol{\theta}) = \sum_{j=1}^n \alpha_j \exp(\boldsymbol{\lambda}^\top \mathbf{f}_j)$ by introducing $\boldsymbol{\lambda} = \boldsymbol{\theta} - \tilde{\boldsymbol{\theta}}$ and $\alpha_j = h_j \exp(\tilde{\boldsymbol{\theta}}^\top \mathbf{f}_j)$. Define the partition function over only the first $i$ components as $Z_i(\boldsymbol{\theta}) = \sum_{j=1}^i \alpha_j \exp(\boldsymbol{\lambda}^\top \mathbf{f}_j)$. When $i = 0$, a trivial quadratic upper bound holds

$$Z_0(\boldsymbol{\theta}) \leq z_0 \exp\left(\tfrac{1}{2}\boldsymbol{\lambda}^\top \boldsymbol{\Sigma}_0 \boldsymbol{\lambda} + \boldsymbol{\lambda}^\top \boldsymbol{\mu}_0\right)$$

with the parameters $z_0 \to 0^+$, $\boldsymbol{\mu}_0 = \mathbf{0}$, and $\boldsymbol{\Sigma}_0 = z_0 \mathbf{I}$. Next, add one term to the current partition function $Z_1(\boldsymbol{\theta}) = Z_0(\boldsymbol{\theta}) + \alpha_1 \exp(\boldsymbol{\lambda}^\top \mathbf{f}_1)$. Apply the current bound $Z_0(\boldsymbol{\theta})$ to obtain

$$Z_1(\boldsymbol{\theta}) \leq z_0 \exp(\tfrac{1}{2}\boldsymbol{\lambda}^\top \boldsymbol{\Sigma}_0 \boldsymbol{\lambda} + \boldsymbol{\lambda}^\top \boldsymbol{\mu}_0) + \alpha_1 \exp(\boldsymbol{\lambda}^\top \mathbf{f}_1).$$

Consider the following change of variables

$$\mathbf{u} = \boldsymbol{\Sigma}_0^{1/2} \boldsymbol{\lambda} - \boldsymbol{\Sigma}_0^{-1/2}(\mathbf{f}_1 - \boldsymbol{\mu}_0))$$
$$\gamma = \tfrac{\alpha_1}{z_0} \exp(\tfrac{1}{2}(\mathbf{f}_1 - \boldsymbol{\mu}_0)^\top \boldsymbol{\Sigma}_0^{-1}(\mathbf{f}_1 - \boldsymbol{\mu}_0))$$

and rewrite the logarithm of the bound as

$$\log Z_1(\boldsymbol{\theta}) \leq \log z_0 - \tfrac{1}{2}(\mathbf{f}_1 - \boldsymbol{\mu}_0)^\top \boldsymbol{\Sigma}_0^{-1}(\mathbf{f}_1 - \boldsymbol{\mu}_0) + \boldsymbol{\lambda}^\top \mathbf{f}_1 + \log\left(\exp(\tfrac{1}{2}\|\mathbf{u}\|^2) + \gamma\right).$$

Apply Lemma 1 (cf. [32] p. 100) to the last term to get

$$\log Z_1(\boldsymbol{\theta}) \leq \log z_0 - \tfrac{1}{2}(\mathbf{f}_1 - \boldsymbol{\mu}_0)^\top \boldsymbol{\Sigma}_0^{-1}(\mathbf{f}_1 - \boldsymbol{\mu}_0) + \boldsymbol{\lambda}^\top \mathbf{f}_1 + \log\left(\exp(\tfrac{1}{2}\|\mathbf{v}\|^2) + \gamma\right)$$
$$+ \frac{\mathbf{v}^\top(\mathbf{u} - \mathbf{v})}{1 + \gamma \exp(-\tfrac{1}{2}\|\mathbf{v}\|^2)} + \tfrac{1}{2}(\mathbf{u} - \mathbf{v})^\top \left(I + \Gamma \mathbf{v}\mathbf{v}^\top\right)(\mathbf{u} - \mathbf{v})$$

where $\Gamma = \frac{\tanh(\frac{1}{2}\log(\gamma\exp(-\frac{1}{2}\|\mathbf{v}\|^2)))}{2\log(\gamma\exp(-\frac{1}{2}\|\mathbf{v}\|^2))}$. The bound in [32] is tight when $\mathbf{u} = \mathbf{v}$. To achieve tightness when $\boldsymbol{\theta} = \tilde{\boldsymbol{\theta}}$ or, equivalently, $\boldsymbol{\lambda} = \mathbf{0}$, we choose $\mathbf{v} = \boldsymbol{\Sigma}_0^{-1/2}(\boldsymbol{\mu}_0 - \mathbf{f}_1)$ yielding

$$Z_1(\boldsymbol{\theta}) \quad \leq \quad z_1\exp\left(\tfrac{1}{2}\boldsymbol{\lambda}^\top\boldsymbol{\Sigma}_1\boldsymbol{\lambda} + \boldsymbol{\lambda}^\top\boldsymbol{\mu}_1\right)$$

where we have

$$
\begin{aligned}
z_1 &= z_0 + \alpha_1 \\
\boldsymbol{\mu}_1 &= \frac{z_0}{z_0 + \alpha_1}\boldsymbol{\mu}_0 + \frac{\alpha_1}{z_0 + \alpha_1}\mathbf{f}_1 \\
\boldsymbol{\Sigma}_1 &= \boldsymbol{\Sigma}_0 + \frac{\tanh(\frac{1}{2}\log(\alpha_1/z_0))}{2\log(\alpha_1/z_0)}(\boldsymbol{\mu}_0 - \mathbf{f}_1)(\boldsymbol{\mu}_0 - \mathbf{f}_1)^\top.
\end{aligned}
$$

This rule updates the bound parameters $z_0, \boldsymbol{\mu}_0, \boldsymbol{\Sigma}_0$ to incorporate an extra term in the sum over $i$ in $Z(\boldsymbol{\theta})$. The process is iterated $n$ times (replacing 1 with $i$ and 0 with $i-1$) to produce an overall bound on all terms.

**Lemma 1** *(See [32] p. 100)*
*For all $\mathbf{u} \in \mathbb{R}^d$, any $\mathbf{v} \in \mathbb{R}^d$ and any $\gamma \geq 0$, the bound* $\log\left(\exp\left(\frac{1}{2}\|\mathbf{u}\|^2\right) + \gamma\right) \leq$

$$\log\left(\exp\left(\tfrac{1}{2}\|\mathbf{v}\|^2\right) + \gamma\right) + \frac{\mathbf{v}^\top(\mathbf{u}-\mathbf{v})}{1 + \gamma\exp(-\frac{1}{2}\|\mathbf{v}\|^2)} + \frac{1}{2}(\mathbf{u}-\mathbf{v})^\top\left(I + \Gamma\mathbf{v}\mathbf{v}^\top\right)(\mathbf{u}-\mathbf{v})$$

*holds when the scalar term* $\Gamma = \frac{\tanh(\frac{1}{2}\log(\gamma\exp(-\|\mathbf{v}\|^2/2)))}{2\log(\gamma\exp(-\|\mathbf{v}\|^2/2))}$. *Equality is achieved when* $\mathbf{u} = \mathbf{v}$.

**Proof of Lemma 1** The proof is provided in [32].

## Supplement for Section 3

**Maximum entropy problem** We show here that partition functions arise naturally in maximum entropy estimation or minimum relative entropy $\mathcal{RE}(p\|h) = \sum_y p(y)\log\frac{p(y)}{h(y)}$ estimation. Consider the following problem:

$$\min_p \mathcal{RE}(p\|h) \text{ s.t. } \sum_y p(y)\mathbf{f}(y) = \mathbf{0}, \sum_y p(y)\mathbf{g}(y) \geq \mathbf{0}.$$

Here, assume that $\mathbf{f} : \Omega \mapsto \mathbb{R}^d$ and $\mathbf{g} : \Omega \mapsto \mathbb{R}^{d'}$ are arbitrary (non-constant) vector-valued functions over the sample space. The solution distribution $p(y) = h(y)\exp\left(\boldsymbol{\theta}^\top\mathbf{f}(y) + \boldsymbol{\vartheta}^\top\mathbf{g}(y)\right)/Z(\boldsymbol{\theta}, \boldsymbol{\vartheta})$ is recovered by the dual optimization

$$\boldsymbol{\theta}, \boldsymbol{\vartheta} = \operatorname*{arg\,max}_{\boldsymbol{\vartheta}\geq\mathbf{0}, \boldsymbol{\theta}} -\log\sum_y h(y)\exp\left(\boldsymbol{\theta}^\top\mathbf{f}(y) + \boldsymbol{\vartheta}^\top\mathbf{g}(y)\right)$$

where $\boldsymbol{\theta} \in \mathbb{R}^d$ and $\boldsymbol{\vartheta} \in \mathbb{R}^{d'}$. These are obtained by minimizing $Z(\boldsymbol{\theta}, \boldsymbol{\vartheta})$ or equivalently by maximizing its negative logarithm. Algorithm 1 permits variational maximization of the dual via the quadratic program

$$\min_{\boldsymbol{\vartheta}\geq\mathbf{0}, \boldsymbol{\theta}} \tfrac{1}{2}(\boldsymbol{\beta} - \tilde{\boldsymbol{\beta}})^\top\boldsymbol{\Sigma}(\boldsymbol{\beta} - \tilde{\boldsymbol{\beta}}) + \boldsymbol{\beta}^\top\boldsymbol{\mu}$$

where $\boldsymbol{\beta}^\top = [\boldsymbol{\theta}^\top\boldsymbol{\vartheta}^\top]$. Note that any general convex hull of constraints $\boldsymbol{\beta} \in \boldsymbol{\Lambda} \subseteq \mathbb{R}^{d+d'}$ could be imposed without loss of generality.

**Proof of Theorem 2** We begin by proving a lemma that will be useful later.

**Lemma 2** *If $\kappa\boldsymbol{\Psi} \succeq \boldsymbol{\Phi} \succ \mathbf{0}$ for $\boldsymbol{\Phi}, \boldsymbol{\Psi} \in \mathbb{R}^{d\times d}$, then*

$$L(\boldsymbol{\theta}) = -\tfrac{1}{2}(\boldsymbol{\theta} - \tilde{\boldsymbol{\theta}})^\top\boldsymbol{\Phi}(\boldsymbol{\theta} - \tilde{\boldsymbol{\theta}}) - (\boldsymbol{\theta} - \tilde{\boldsymbol{\theta}})^\top\boldsymbol{\mu}$$
$$U(\boldsymbol{\theta}) = -\tfrac{1}{2}(\boldsymbol{\theta} - \tilde{\boldsymbol{\theta}})^\top\boldsymbol{\Psi}(\boldsymbol{\theta} - \tilde{\boldsymbol{\theta}}) - (\boldsymbol{\theta} - \tilde{\boldsymbol{\theta}})^\top\boldsymbol{\mu}$$

*satisfy $\sup_{\boldsymbol{\theta}\in\boldsymbol{\Lambda}} L(\boldsymbol{\theta}) \geq \frac{1}{\kappa}\sup_{\boldsymbol{\theta}\in\boldsymbol{\Lambda}} U(\boldsymbol{\theta})$ for any convex $\boldsymbol{\Lambda} \subseteq \mathbb{R}^d$, $\tilde{\boldsymbol{\theta}} \in \boldsymbol{\Lambda}$, $\boldsymbol{\mu} \in \mathbb{R}^d$ and $\kappa \in \mathbb{R}^+$.*

**Proof of Lemma 2** Define the primal problems of interest as $\mathbf{P}_L = \sup_{\boldsymbol{\theta} \in \boldsymbol{\Lambda}} L(\boldsymbol{\theta})$ and $\mathbf{P}_U = \sup_{\boldsymbol{\theta} \in \boldsymbol{\Lambda}} U(\boldsymbol{\theta})$. The constraints $\boldsymbol{\theta} \in \boldsymbol{\Lambda}$ can be summarized by a set of linear inequalities $\mathbf{A}\boldsymbol{\theta} \leq \mathbf{b}$ where $\mathbf{A} \in \mathbb{R}^{k \times d}$ and $\mathbf{b} \in \mathbb{R}^k$ for some (possibly infinite) $k \in \mathbb{Z}$. Apply the change of variables $\mathbf{z} = \boldsymbol{\theta} - \tilde{\boldsymbol{\theta}}$. The constraint $\mathbf{A}(\mathbf{z}+\tilde{\boldsymbol{\theta}}) \leq \mathbf{b}$ simplifies into $\mathbf{A}\mathbf{z} \leq \tilde{\mathbf{b}}$ where $\tilde{\mathbf{b}} = \mathbf{b} - \mathbf{A}\tilde{\boldsymbol{\theta}}$. Since $\tilde{\boldsymbol{\theta}} \in \boldsymbol{\Lambda}$, it is easy to show that $\tilde{\mathbf{b}} \geq \mathbf{0}$. We obtain the equivalent primal problems $\mathbf{P}_L = \sup_{\mathbf{A}\mathbf{z} \leq \tilde{\mathbf{b}}} -\frac{1}{2}\mathbf{z}^\top \boldsymbol{\Phi} \mathbf{z} - \mathbf{z}^\top \boldsymbol{\mu}$ and $\mathbf{P}_U = \sup_{\mathbf{A}\mathbf{z} \leq \tilde{\mathbf{b}}} -\frac{1}{2}\mathbf{z}^\top \boldsymbol{\Psi} \mathbf{z} - \mathbf{z}^\top \boldsymbol{\mu}$. The corresponding dual problems are

$$\mathbf{D}_L = \inf_{\mathbf{y} \geq \mathbf{0}} \frac{\mathbf{y}^\top \mathbf{A}\boldsymbol{\Phi}^{-1}\mathbf{A}^\top \mathbf{y}}{2} + \mathbf{y}^\top \mathbf{A}\boldsymbol{\Phi}^{-1}\boldsymbol{\mu} + \mathbf{y}^\top \tilde{\mathbf{b}} + \frac{\boldsymbol{\mu}^\top \boldsymbol{\Phi}^{-1}\boldsymbol{\mu}}{2}$$

$$\mathbf{D}_U = \inf_{\mathbf{y} \geq \mathbf{0}} \frac{\mathbf{y}^\top \mathbf{A}\boldsymbol{\Psi}^{-1}\mathbf{A}^\top \mathbf{y}}{2} + \mathbf{y}^\top \mathbf{A}\boldsymbol{\Psi}^{-1}\boldsymbol{\mu} + \mathbf{y}^\top \tilde{\mathbf{b}} + \frac{\boldsymbol{\mu}^\top \boldsymbol{\Psi}^{-1}\boldsymbol{\mu}}{2}.$$

Due to strong duality, $\mathbf{P}_L = \mathbf{D}_L$ and $\mathbf{P}_U = \mathbf{D}_U$. Apply the inequalities $\boldsymbol{\Phi} \preceq \kappa \boldsymbol{\Psi}$ and $\mathbf{y}^\top \tilde{\mathbf{b}} > 0$ as

$$\mathbf{P}_L \geq \sup_{\mathbf{A}\mathbf{z} \leq \tilde{\mathbf{b}}} -\frac{\kappa}{2}\mathbf{z}^\top \boldsymbol{\Psi} \mathbf{z} - \mathbf{z}^\top \boldsymbol{\mu} = \inf_{\mathbf{y} \geq \mathbf{0}} \frac{\mathbf{y}^\top \mathbf{A}\boldsymbol{\Psi}^{-1}\mathbf{A}^\top \mathbf{y}}{2\kappa} + \frac{\mathbf{y}^\top \mathbf{A}\boldsymbol{\Psi}^{-1}\boldsymbol{\mu}}{\kappa} + \mathbf{y}^\top \tilde{\mathbf{b}} + \frac{\boldsymbol{\mu}^\top \boldsymbol{\Psi}^{-1}\boldsymbol{\mu}}{2\kappa}$$

$$\geq \frac{1}{\kappa}\mathbf{D}_U = \frac{1}{\kappa}\mathbf{P}_U.$$

This proves that $\mathbf{P}_L \geq \frac{1}{\kappa}\mathbf{P}_U$.

We will use the above to prove Theorem 2. First, we will upper-bound (in the Loewner ordering sense) the matrices $\boldsymbol{\Sigma}_j$ in Algorithm 2. Since $\|\mathbf{f}_{x_j}(y)\|^2 \leq r$ for all $y \in \Omega_j$ and since $\boldsymbol{\mu}_j$ in Algorithm 1 is a convex combination of $\mathbf{f}_{x_j}(y)$, the outer-product terms in the update for $\boldsymbol{\Sigma}_j$ satisfy

$$(\mathbf{f}_{x_j}(y) - \boldsymbol{\mu})(\mathbf{f}_{x_j}(y) - \boldsymbol{\mu})^\top \preceq 4r^2 \mathbf{I}.$$

Thus, $\boldsymbol{\Sigma}_j \preceq \mathcal{F}(\alpha_1, \dots, \alpha_n)4r^2 \mathbf{I}$ holds where

$$\mathcal{F}(\alpha_1, \dots, \alpha_n) = \sum_{i=2}^{n} \frac{\tanh(\frac{1}{2}\log(\frac{\alpha_i}{\sum_{k=1}^{i-1}\alpha_k}))}{2\log(\frac{\alpha_i}{\sum_{k=1}^{i-1}\alpha_k})}$$

using the definition of $\alpha_1, \dots, \alpha_n$ in the proof of Theorem 1. The formula for $\mathcal{F}$ starts at $i = 2$ since $z_0 \to 0^+$. Assume permutation $\pi$ is sampled uniformly at random. The expected value of $\mathcal{F}$ is then

$$\mathrm{E}_\pi[\mathcal{F}(\alpha_1, \dots, \alpha_n)] = \frac{1}{n!}\sum_{\pi}\sum_{i=2}^{n} \frac{\tanh(\frac{1}{2}\log(\frac{\alpha_{\pi(i)}}{\sum_{k=1}^{i-1}\alpha_{\pi(k)}}))}{2\log(\frac{\alpha_{\pi(i)}}{\sum_{k=1}^{i-1}\alpha_{\pi(k)}})}.$$

We claim that the expectation is maximized when all $\alpha_i = 1$ or any positive constant. Also, $\mathcal{F}$ is invariant under uniform scaling of its arguments. Write the expected value of $\mathcal{F}$ as $\mathrm{E}$ for short. Next, consider $\frac{\partial \mathrm{E}}{\partial \alpha_l}$ at the setting $\alpha_i = 1, \forall i$. Due to the expectation over $\pi$, we have $\frac{\partial \mathrm{E}}{\partial \alpha_l} = \frac{\partial \mathrm{E}}{\partial \alpha_o}$ for any $l, o$. Therefore, the gradient vector is constant when all $\alpha_i = 1$. Since $\mathcal{F}(\alpha_1, \dots, \alpha_n)$ is invariant to scaling, the gradient vector must therefore be the all zeros vector. Thus, the point when all $\alpha_i = 1$ is an extremum or a saddle. Next, consider $\frac{\partial}{\partial \alpha_o}\frac{\partial \mathrm{E}}{\partial \alpha_l}$ for any $l, o$. At the setting $\alpha_i = 1$, $\frac{\partial^2 \mathrm{E}}{\partial \alpha_l^2} = -c(n)$ and, $\frac{\partial}{\partial \alpha_o}\frac{\partial \mathrm{E}}{\partial \alpha_l} = c(n)/(n-1)$ for some non-negative constant function $c(n)$. Thus, the $\alpha_i = 1$ extremum is locally concave and is a maximum. This establishes that $\mathrm{E}_\pi[\mathcal{F}(\alpha_1, \dots, \alpha_n)] \leq \mathrm{E}_\pi[\mathcal{F}(1, \dots, 1)]$ and yields the Loewner bound

$$\boldsymbol{\Sigma}_j \preceq \left(2r^2 \sum_{i=1}^{n-1} \frac{\tanh(\log(i)/2)}{\log(i)}\right)\mathbf{I} = \omega \mathbf{I}.$$

Apply this bound to each $\boldsymbol{\Sigma}_j$ in the lower bound on $J(\boldsymbol{\theta})$ and also note a corresponding upper bound

$$J(\boldsymbol{\theta}) \geq J(\tilde{\boldsymbol{\theta}}) - \frac{t\omega + t\lambda}{2}\|\boldsymbol{\theta} - \tilde{\boldsymbol{\theta}}\|^2 - \sum_{j}(\boldsymbol{\theta} - \tilde{\boldsymbol{\theta}})^\top(\boldsymbol{\mu}_j - \mathbf{f}_{x_j}(y_j))$$

$$J(\boldsymbol{\theta}) \leq J(\tilde{\boldsymbol{\theta}}) - \frac{t\lambda}{2}\|\boldsymbol{\theta} - \tilde{\boldsymbol{\theta}}\|^2 - \sum_{j}(\boldsymbol{\theta} - \tilde{\boldsymbol{\theta}})^\top(\boldsymbol{\mu}_j - \mathbf{f}_{x_j}(y_j))$$

which follows from Jensen's inequality. Define the current $\tilde{\boldsymbol{\theta}}$ at time $\tau$ as $\boldsymbol{\theta}_\tau$ and denote by $L_\tau(\boldsymbol{\theta})$ the above lower bound and by $U_\tau(\boldsymbol{\theta})$ the above upper bound at time $\tau$. Clearly, $L_\tau(\boldsymbol{\theta}) \leq J(\boldsymbol{\theta}) \leq U_\tau(\boldsymbol{\theta})$ with equality when $\boldsymbol{\theta} = \boldsymbol{\theta}_\tau$. Algorithm 2 maximizes $J(\boldsymbol{\theta})$ after initializing at $\boldsymbol{\theta}_0$ and performing an update by maximizing a lower bound based on $\boldsymbol{\Sigma}_j$. Since $L_\tau(\boldsymbol{\theta})$ replaces the definition of $\boldsymbol{\Sigma}_j$ with $\omega \mathbf{I} \succeq \boldsymbol{\Sigma}_j$, $L_\tau(\boldsymbol{\theta})$ is a looser bound than the one used by Algorithm 2. Thus, performing $\boldsymbol{\theta}_{\tau+1} = \arg\max_{\boldsymbol{\theta} \in \boldsymbol{\Lambda}} L_\tau(\boldsymbol{\theta})$ makes less progress than a step of Algorithm 1. Consider computing the slower update at each iteration $\tau$ and returning $\boldsymbol{\theta}_{\tau+1} = \arg\max_{\boldsymbol{\theta} \in \boldsymbol{\Lambda}} L_\tau(\boldsymbol{\theta})$. Setting $\boldsymbol{\Phi} = (t\omega + t\lambda)\mathbf{I}$, $\boldsymbol{\Psi} = t\lambda \mathbf{I}$ and $\kappa = \frac{\omega + \lambda}{\lambda}$ allows us to apply Lemma 2 as follows

$$\sup_{\boldsymbol{\theta} \in \boldsymbol{\Lambda}} L_\tau(\boldsymbol{\theta}) - L_\tau(\boldsymbol{\theta}_\tau) = \frac{1}{\kappa} \sup_{\boldsymbol{\theta} \in \boldsymbol{\Lambda}} U_\tau(\boldsymbol{\theta}) - U_\tau(\boldsymbol{\theta}_\tau).$$

Since $L_\tau(\boldsymbol{\theta}_\tau) = J(\boldsymbol{\theta}_\tau) = U_\tau(\boldsymbol{\theta}_\tau)$, $J(\boldsymbol{\theta}_{\tau+1}) \geq \sup_{\boldsymbol{\theta} \in \boldsymbol{\Lambda}} L_\tau(\boldsymbol{\theta})$ and $\sup_{\boldsymbol{\theta} \in \boldsymbol{\Lambda}} U_\tau(\boldsymbol{\theta}) \geq J(\boldsymbol{\theta}^*)$, we obtain

$$J(\boldsymbol{\theta}_{\tau+1}) - J(\boldsymbol{\theta}^*) \geq \left(1 - \frac{1}{\kappa}\right)(J(\boldsymbol{\theta}_\tau) - J(\boldsymbol{\theta}^*)).$$

Iterate the above inequality starting at $t = 0$ to obtain

$$J(\boldsymbol{\theta}_\tau) - J(\boldsymbol{\theta}^*) \geq \left(1 - \frac{1}{\kappa}\right)^\tau (J(\boldsymbol{\theta}_0) - J(\boldsymbol{\theta}^*)).$$

A solution within a multiplicative factor of $\epsilon$ implies that $\epsilon = \left(1 - \frac{1}{\kappa}\right)^\tau$ or $\log(1/\epsilon) = \tau \log \frac{\kappa}{\kappa-1}$. Inserting the definition for $\kappa$ shows that the number of iterations $\tau$ is at most $\left\lceil \frac{\log(1/\epsilon)}{\log \frac{\kappa}{\kappa-1}} \right\rceil$ or $\left\lceil \frac{\log(1/\epsilon)}{\log(1+\lambda/\omega)} \right\rceil$. Inserting the definition for $\omega$ gives the bound.

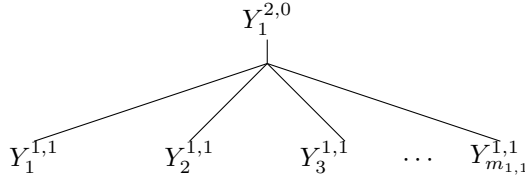

Figure 3: Junction tree of depth 2.

**Algorithm 5** SmallJunctionTree

Input Parameters $\tilde{\boldsymbol{\theta}}$ and $h(u), f(u) \ \forall u \in Y_1^{2,0}$ and $z_i, \boldsymbol{\Sigma}_i, \boldsymbol{\mu}_i \ \forall i = 1, \ldots, m_{1,1}$

Initialize $z \to 0^+, \boldsymbol{\mu} = \mathbf{0}, \boldsymbol{\Sigma} = z\mathbf{I}$

For each configuration $u \in Y_1^{2,0}$ {

$\alpha \quad = h(u)(\prod_{i=1}^{m_{1,1}} z_i \exp(-\tilde{\boldsymbol{\theta}}^\top \boldsymbol{\mu}_i)) \exp(\tilde{\boldsymbol{\theta}}^\top (f(u) + \sum_{i=1}^{m_{1,1}} \boldsymbol{\mu}_i)) = h(u) \exp(\tilde{\boldsymbol{\theta}}^\top f(u)) \prod_{i=1}^{m_{1,1}} z_i$

$\mathbf{l} \quad = f(u) + \sum_{i=1}^{m_{1,1}} \boldsymbol{\mu}_i - \boldsymbol{\mu}$

$\boldsymbol{\Sigma} += \sum_{i=1}^{m_{1,1}} \boldsymbol{\Sigma}_i + \frac{\tanh(\frac{1}{2}\log(\alpha/z))}{2\log(\alpha/z)} \mathbf{l}\mathbf{l}^\top$

$\boldsymbol{\mu} += \frac{\alpha}{z+\alpha}\mathbf{l}$

$z \ += \alpha \qquad$ }

Output $z, \boldsymbol{\mu}, \boldsymbol{\Sigma}$

## Supplement for Section 5

**Proof of correctness for Algorithm 3** Consider a simple junction tree of depth 2 shown on Figure 3. The notation $Y_c^{a,b}$ refers to the $c^{th}$ tree node located at tree level $a$ (first level is considered as the one with tree leaves) whose parent is the $b^{th}$ from the higher tree level (the root has no parent so $b = 0$). Let $\sum_{Y_{c_1}^{a_1,b_1}}$ refer to the sum over all configurations of variables in $Y_{c_1}^{a_1,b_1}$ and $\sum_{Y_{c_1}^{a_1,b_1} \setminus Y_{c_2}^{a_2,b_2}}$ refers to the sum over all configurations of variables that are in $Y_{c_1}^{a_1,b_1}$ but not in $Y_{c_2}^{a_2,b_2}$. Let $m_{a,b}$ denote the number of children of the $b^{th}$ node located at tree level $a + 1$. For short-hand, use $\psi(Y) = h(Y) \exp(\boldsymbol{\theta}^\top f(Y))$. The partition function can be expressed as:

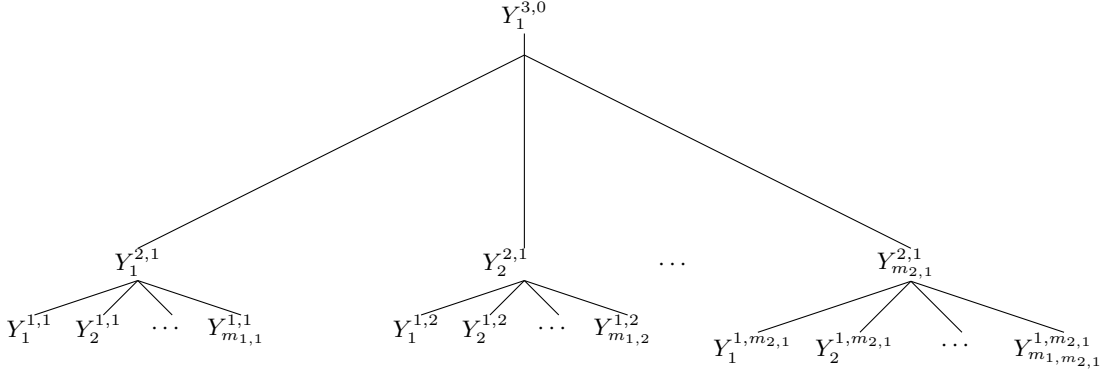

Figure 4: Junction tree of depth 3.

$$Z(\boldsymbol{\theta}) = \sum_{u \in Y_1^{2,0}} \left[ \psi(u) \prod_{i=1}^{m_{1,1}} \left( \sum_{v \in Y_i^{1,1} \setminus Y_1^{2,0}} \psi(v) \right) \right]$$

$$\leq \sum_{u \in Y_1^{2,0}} \left[ \psi(u) \prod_{i=1}^{m_{1,1}} z_i \exp\left( \frac{1}{2} \left( \boldsymbol{\theta} - \tilde{\boldsymbol{\theta}} \right)^\top \boldsymbol{\Sigma}_i (\boldsymbol{\theta} - \tilde{\boldsymbol{\theta}}) + (\boldsymbol{\theta} - \tilde{\boldsymbol{\theta}})^\top \boldsymbol{\mu}_i \right) \right]$$

$$= \sum_{u \in Y_1^{2,0}} \left[ h(u) \exp(\boldsymbol{\theta}^\top f(u)) \prod_{i=1}^{m_{1,1}} z_i \exp\left( \frac{1}{2} (\boldsymbol{\theta} - \tilde{\boldsymbol{\theta}})^\top \boldsymbol{\Sigma}_i (\boldsymbol{\theta} - \tilde{\boldsymbol{\theta}}) + (\boldsymbol{\theta} - \tilde{\boldsymbol{\theta}})^\top \boldsymbol{\mu}_i \right) \right]$$

where the upper-bound is obtained by applying Theorem 1 to each of the terms $\sum_{v \in Y_i^{1,1} \setminus Y_1^{2,0}} \psi(v)$. By simply rearranging terms we get:

$$Z(\boldsymbol{\theta}) \leq \sum_{u \in Y_1^{2,0}} \left[ h(u) \left( \prod_{i=1}^{m_{1,1}} z_i \exp(-\tilde{\boldsymbol{\theta}}^\top \boldsymbol{\mu}_i) \right) \exp\left( \boldsymbol{\theta}^\top \left( f(u) + \sum_{i=1}^{m_{1,1}} \boldsymbol{\mu}_i \right) \right) \right.$$

$$\left. \exp\left( \frac{1}{2} (\boldsymbol{\theta} - \tilde{\boldsymbol{\theta}})^\top \left( \sum_{i=1}^{m_{1,1}} \boldsymbol{\Sigma}_i \right) (\boldsymbol{\theta} - \tilde{\boldsymbol{\theta}}) \right) \right].$$

One can prove that this expression can be upper-bounded by $z \exp\left( \frac{1}{2} (\boldsymbol{\theta} - \hat{\boldsymbol{\theta}})^\top \boldsymbol{\Sigma} (\boldsymbol{\theta} - \hat{\boldsymbol{\theta}}) + (\boldsymbol{\theta} - \hat{\boldsymbol{\theta}})^\top \boldsymbol{\mu} \right)$ where $z$, $\boldsymbol{\Sigma}$ and $\boldsymbol{\mu}$ can be computed using Algorithm 5 (a simplification of Algorithm 3). We will call this result Lemma A. The proof is similar to the proof of Theorem 1 so is not repeated here.

Consider enlarging the tree to a depth 3 as shown on Figure 4. The partition function is now

$$Z(\boldsymbol{\theta}) = \sum_{u \in Y_1^{3,0}} \left[ \psi(u) \prod_{i=1}^{m_{2,1}} \left( \sum_{v \in Y_i^{2,1} \setminus Y_1^{3,0}} \left( \psi(v) \prod_{j=1}^{m_{1,i}} \left( \sum_{w \in Y_j^{1,i} \setminus Y_i^{2,1}} \psi(w) \right) \right) \right) \right].$$

By Lemma A we can upper bound each $\sum_{v \in Y_i^{2,1} \setminus Y_1^{3,0}} \left( \psi(v) \prod_{j=1}^{m_{1,i}} \left( \sum_{w \in Y_j^{1,i} \setminus Y_i^{2,1}} \psi(w) \right) \right)$ term by the expression $z_i \exp\left( \frac{1}{2} (\boldsymbol{\theta} - \hat{\boldsymbol{\theta}})^\top \boldsymbol{\Sigma}_i (\boldsymbol{\theta} - \hat{\boldsymbol{\theta}}) + (\boldsymbol{\theta} - \hat{\boldsymbol{\theta}})^\top \boldsymbol{\mu}_i \right)$. This yields

$$Z(\boldsymbol{\theta}) \leq \sum_{u \in Y_1^{3,0}} \left[ \psi(u) \prod_{i=1}^{m_{2,1}} z_i \exp\left( \frac{1}{2} (\boldsymbol{\theta} - \tilde{\boldsymbol{\theta}})^\top \boldsymbol{\Sigma}_i (\boldsymbol{\theta} - \tilde{\boldsymbol{\theta}}) + (\boldsymbol{\theta} - \tilde{\boldsymbol{\theta}})^\top \boldsymbol{\mu}_i \right) \right].$$

This process can be viewed as collapsing the sub-trees $S_1^{2,1}$, $S_2^{2,1}$, ..., $S_{m_{2,1}}^{2,1}$ to super-nodes that are represented by bound parameters, $z_i$, $\boldsymbol{\Sigma}_i$ and $\boldsymbol{\mu}_i, i = \{1, 2, \cdots, m_{2,1}\}$, where the sub-trees are

defined as:

$$
\begin{aligned}
S_1^{2,1} &= \{Y_1^{2,1}, Y_1^{1,1}, Y_2^{1,1}, Y_3^{1,1}, \ldots, Y_{m_{1,1}}^{1,1}\} \\
S_2^{2,1} &= \{Y_2^{2,1}, Y_1^{1,2}, Y_2^{1,2}, Y_3^{1,2}, \ldots, Y_{m_{1,2}}^{1,2}\} \\
&\vdots \\
S_{m_{2,1}}^{2,1} &= \{Y_{m_{2,1}}^{2,1}, Y_1^{1,m_{2,1}}, Y_2^{1,m_{2,1}}, Y_3^{1,m_{2,1}}, \ldots, Y_{m_{1,m_{2,1}}}^{1,m_{2,1}}\}.
\end{aligned}
$$

Notice that the obtained expression can be further upper bounded using again Lemma A (induction) yielding a bound of the form: $z \exp\left(\frac{1}{2}(\boldsymbol{\theta} - \hat{\boldsymbol{\theta}})^\top \boldsymbol{\Sigma}(\boldsymbol{\theta} - \hat{\boldsymbol{\theta}}) + (\boldsymbol{\theta} - \hat{\boldsymbol{\theta}})^\top \boldsymbol{\mu}\right)$.

Finally, for a general tree, follow the same steps described above, starting from leaves and collapsing nodes to super-nodes, each represented by bound parameters. This procedure effectively yields Algorithm 3 for the junction tree under consideration.

## Supplement for Section 6

**Proof of correctness for Algorithm 4** We begin by proving a lemma that will be useful later.

**Lemma 3** *For all* $\mathbf{x} \in \mathbb{R}^d$ *and for all* $\mathbf{l} \in \mathbb{R}^d$,

$$
\sum_{i=1}^d \mathbf{x}(i)^2 \mathbf{l}(i)^2 \geq \left( \sum_{i=1}^d \mathbf{x}(i) \frac{\mathbf{l}(i)^2}{\sqrt{\sum_{j=1}^d \mathbf{l}(j)^2}} \right)^2.
$$

**Proof of Lemma 3** By Jensen's inequality,

$$
\sum_{i=1}^d \mathbf{x}(i)^2 \frac{\mathbf{l}(i)^2}{\sum_{j=1}^d \mathbf{l}(j)^2} \geq \left( \sum_{i=1}^d \frac{\mathbf{x}(i)\mathbf{l}(i)^2}{\sum_{j=1}^d \mathbf{l}(j)^2} \right)^2 \iff \sum_{i=1}^d \mathbf{x}(i)^2 l(i)^2 \geq \left( \sum_{i=1}^d \frac{\mathbf{x}(i)\mathbf{l}(i)^2}{\sqrt{\sum_{j=1}^d \mathbf{l}(j)^2}} \right)^2.
$$

Now we prove the correctness of Algorithm 4. At the $i^{th}$ iteration, the algorithm stores $\boldsymbol{\Sigma}_i$ using a low-rank representation $\mathbf{V}_i^\top \mathbf{S}_i \mathbf{V}_i + \mathbf{D}_i$ where $\mathbf{V}_i \in \mathbb{R}^{k \times d}$ is orthonormal, $\mathbf{S}_i \in \mathbb{R}^{k \times k}$ positive semi-definite and $\mathbf{D}_i \in \mathbb{R}^{d \times d}$ is non-negative diagonal. The diagonal terms $\mathbf{D}$ are initialized to $t\lambda\mathbf{I}$ where $\lambda$ is the regularization term. To mimic Algorithm 1 we must increment the $\boldsymbol{\Sigma}$ matrix by a rank one update of the form $\boldsymbol{\Sigma}_i = \boldsymbol{\Sigma}_{i-1} + \mathbf{r}_i \mathbf{r}_i^\top$. By projecting $\mathbf{r}_i$ onto each eigenvector in $\mathbf{V}$, we can decompose it as $\mathbf{r}_i = \sum_{j=1}^k \mathbf{V}_{i-1}(j,\cdot)\mathbf{r}_i \mathbf{V}_{i-1}(j,\cdot)^\top + \mathbf{g} = \mathbf{V}_{i-1}^\top \mathbf{V}_{i-1}\mathbf{r}_i + \mathbf{g}$ where $\mathbf{g}$ is the remaining residue. Thus the update rule can be rewritten as:

$$
\begin{aligned}
\boldsymbol{\Sigma}_i &= \boldsymbol{\Sigma}_{i-1} + \mathbf{r}_i \mathbf{r}_i^\top = \mathbf{V}_{i-1}^\top \mathbf{S}_{i-1} \mathbf{V}_{i-1} + \mathbf{D}_{i-1} + (\mathbf{V}_{i-1}^\top \mathbf{V}_{i-1}\mathbf{r}_i + \mathbf{g})(\mathbf{V}_{i-1}^\top \mathbf{V}_{i-1}\mathbf{r}_i + \mathbf{g})^\top \\
&= \mathbf{V}_{i-1}^\top (\mathbf{S}_{i-1} + \mathbf{V}_{i-1}\mathbf{r}_i \mathbf{r}_i^\top \mathbf{V}_{i-1}^\top)\mathbf{V}_{i-1} + \mathbf{D}_{i-1} + \mathbf{g}\mathbf{g}^\top = \mathbf{V}_{i-1}'^\top \mathbf{S}_{i-1}' \mathbf{V}_{i-1}' + \mathbf{g}\mathbf{g}^\top + \mathbf{D}_{i-1}
\end{aligned}
$$

where we define $\mathbf{V}_{i-1}' = \mathbf{Q}_{i-1}\mathbf{V}_{i-1}$ and defined $\mathbf{Q}_{i-1}$ in terms of the singular value decomposition, $\mathbf{Q}_{i-1}^\top \mathbf{S}_{i-1}' \mathbf{Q}_{i-1} = \mathrm{svd}(\mathbf{S}_{i-1} + \mathbf{V}_{i-1}\mathbf{r}_i \mathbf{r}_i^\top \mathbf{V}_{i-1}^\top)$. Note that $\mathbf{S}_{i-1}'$ is diagonal and nonnegative by construction. The current formula for $\boldsymbol{\Sigma}_i$ shows that we have a rank $(k+1)$ system (plus diagonal term) which needs to be converted back to a rank $k$ system (plus diagonal term) which we denote by $\boldsymbol{\Sigma}_i'$. We have two options as follows.
Case 1) Remove $\mathbf{g}$ from $\boldsymbol{\Sigma}_i$ to obtain

$$
\boldsymbol{\Sigma}_i' = \mathbf{V}_{i-1}'^\top \mathbf{S}_{i-1}' \mathbf{V}_{i-1}' + \mathbf{D}_{i-1} = \boldsymbol{\Sigma}_i - \mathbf{g}\mathbf{g}^\top = \boldsymbol{\Sigma}_i - c\mathbf{v}\mathbf{v}^\top
$$

where $c = \|\mathbf{g}\|^2$ and $\mathbf{v} = \frac{1}{\|\mathbf{g}\|}\mathbf{g}$.
Case 2) Remove the $m^{th}$ (smallest) eigenvalue in $\mathbf{S}_{i-1}'$ and its corresponding eigenvector:

$$
\boldsymbol{\Sigma}_i' = \mathbf{V}_{i-1}'^\top \mathbf{S}_{i-1}' \mathbf{V}_{i-1}' + \mathbf{D}_{i-1} + \mathbf{g}\mathbf{g}^\top - \mathbf{S}'(m,m)\mathbf{V}'(m,\cdot)^\top \mathbf{V}'(m,\cdot) = \boldsymbol{\Sigma}_i - c\mathbf{v}\mathbf{v}^\top
$$

where $c = \mathbf{S}'(m,m)$ and $\mathbf{v} = \mathbf{V}(m,\cdot)'$.

Clearly, both cases can be written as an update of the form $\boldsymbol{\Sigma}_i' = \boldsymbol{\Sigma}_i + c\mathbf{v}\mathbf{v}^\top$ where $c \geq 0$ and $\mathbf{v}^\top\mathbf{v} = 1$. We choose the case with smaller $c$ value to minimize the change as we drop from a system of order $(k+1)$ to order $k$. Discarding the smallest singular value and its corresponding eigenvector would violate the bound. Instead, consider absorbing this term into the diagonal component to preserve the bound. Formally, we look for a diagonal matrix $\mathbf{F}$ such that $\boldsymbol{\Sigma}_i'' = \boldsymbol{\Sigma}_i' + \mathbf{F}$ which also maintains $\mathbf{x}^\top\boldsymbol{\Sigma}_i''\mathbf{x} \geq \mathbf{x}^\top\boldsymbol{\Sigma}_i\mathbf{x}$ for all $\mathbf{x} \in \mathbb{R}^d$. Thus, we want to satisfy:

$$\mathbf{x}^\top\boldsymbol{\Sigma}_i''\mathbf{x} \geq \mathbf{x}^\top\boldsymbol{\Sigma}_i\mathbf{x} \iff \mathbf{x}^\top c\mathbf{v}\mathbf{v}^\top\mathbf{x} \leq \mathbf{x}^\top\mathbf{F}\mathbf{x} \iff c\left(\sum_{i=1}^d \mathbf{x}(i)\mathbf{v}(i)\right)^2 \leq \sum_{i=1}^d \mathbf{x}(i)^2\mathbf{F}(i)$$

where, for ease of notation, we take $\mathbf{F}(i) = \mathbf{F}(i,i)$.

Define $\mathbf{v}' = \frac{1}{w}\mathbf{v}$ where $w = \mathbf{v}^\top\mathbf{1}$. Consider the case where $\mathbf{v} \geq \mathbf{0}$ though we will soon get rid of this assumption. We need an $\mathbf{F}$ such that $\sum_{i=1}^d \mathbf{x}(i)^2\mathbf{F}(i) \geq c\left(\sum_{i=1}^d \mathbf{x}(i)\mathbf{v}(i)\right)^2$. Equivalently, we need $\sum_{i=1}^d \mathbf{x}(i)^2\frac{\mathbf{F}(i)}{cw^2} \geq \left(\sum_{i=1}^d \mathbf{x}(i)\mathbf{v}(i)'\right)^2$. Define $\mathbf{F}(i)' = \frac{\mathbf{F}(i)}{cw^2}$ for all $i = 1, \ldots, d$. So, we need an $\mathbf{F}'$ such that $\sum_{i=1}^d \mathbf{x}(i)^2\mathbf{F}(i)' \geq \left(\sum_{i=1}^d \mathbf{x}(i)\mathbf{v}(i)'\right)^2$. Using Lemma 3 it is easy to show that we may choose $\mathbf{F}'(i) = \mathbf{v}(i)'$. Thus, we obtain $\mathbf{F}(i) = cw^2\mathbf{F}(i)' = cw\mathbf{v}(i)$. Therefore, for all $\mathbf{x} \in \mathbb{R}^d$, all $\mathbf{v} \geq \mathbf{0}$, and for $\mathbf{F}(i) = c\mathbf{v}(i)\sum_{j=1}^d \mathbf{v}(j)$ we have

$$\sum_{i=1}^d \mathbf{x}(i)^2\mathbf{F}(i) \geq c\left(\sum_{i=1}^d \mathbf{x}(i)\mathbf{v}(i)\right)^2. \tag{3}$$

To generalize the inequality to hold for all vectors $\mathbf{v} \in \mathbb{R}^d$ with potentially negative entries, it is sufficient to set $\mathbf{F}(i) = c|\mathbf{v}(i)|\sum_{j=1}^d |\mathbf{v}(j)|$. To verify this, consider flipping the sign of any $\mathbf{v}(i)$. The left side of the Inequality 3 does not change. For the right side of this inequality, flipping the sign of $\mathbf{v}(i)$ is equivalent to flipping the sign of $\mathbf{x}(i)$ and not changing the sign of $\mathbf{v}(i)$. However, in this case the inequality holds as shown before (it holds for any $\mathbf{x} \in \mathbb{R}^d$). Thus for all $\mathbf{x}, \mathbf{v} \in \mathbb{R}^d$ and for $\mathbf{F}(i) = c|\mathbf{v}(i)|\sum_{j=1}^d |\mathbf{v}(j)|$, Inequality 3 holds.

## Supplement for Section 7

**Small scale experiments** In additional small-scale experiments, we compared Algorithm 2 with steepest descent (SD), conjugate gradient (CG), BFGS and Newton-Raphson. Small-scale problems may be interesting in real-time learning settings, for example, when a website has to learn from a user's uploaded labeled data in a split second to perform real-time retrieval. We considered logistic regression on five UCI data sets where missing values were handled via mean-imputation. A range of regularization settings $\lambda \in \{10^0, 10^2, 10^4\}$ was explored and all algorithms were initialized from the same ten random start-points. Table 3 shows the average number of seconds each algorithm needed to achieve the same solution that BFGS converged to (all algorithms achieve the same solution due to concavity). The bound is the fastest algorithm as indicated in bold.

| $data|\lambda$ | $a|10^0$ | $a|10^2$ | $a|10^4$ | $b|10^0$ | $b|10^2$ | $b|10^4$ | $c|10^0$ | $c|10^2$ | $c|10^4$ | $d|10^0$ | $d|10^2$ | $d|10^4$ | $e|10^0$ | $e|10^2$ | $e|10^4$ |
|---|---|---|---|---|---|---|---|---|---|---|---|---|---|---|---|
| BFGS | 1.90 | 0.89 | 2.45 | 3.14 | 2.00 | 1.60 | 4.09 | 1.03 | 1.90 | 5.62 | 2.88 | 3.28 | 2.63 | 2.01 | 1.49 |
| SD | 1.74 | 0.92 | 1.60 | 2.18 | 6.17 | 5.83 | 1.92 | 0.64 | 0.56 | 12.04 | 1.27 | 1.94 | 2.68 | 2.49 | 1.54 |
| CG | 0.78 | 0.83 | 0.85 | 0.70 | 0.67 | 0.83 | 0.65 | 0.64 | 0.72 | 1.36 | 1.21 | 1.23 | 0.48 | 0.55 | 0.43 |
| Newton | 0.31 | 0.25 | 0.22 | 0.43 | 0.37 | 0.35 | 0.39 | 0.34 | 0.32 | 0.92 | 0.63 | 0.60 | 0.35 | 0.26 | 0.20 |
| Bound | **0.01** | **0.01** | **0.01** | **0.07** | **0.04** | **0.04** | **0.07** | **0.02** | **0.02** | **0.16** | **0.09** | **0.07** | **0.03** | **0.03** | **0.03** |

Table 3: Convergence time in seconds under various regularization levels for a) Bupa ($t = 345, \mathsf{dim} = 7$), b) Wine ($t = 178, \mathsf{dim} = 14$), c) Heart ($t = 187, \mathsf{dim} = 23$), d) Ion ($t = 351, \mathsf{dim} = 34$), and e) Hepatitis ($t = 155, \mathsf{dim} = 20$) data sets.

**Influence of rank $k$ on bound performance in large scale experiments** We also examined the influence of $k$ on bound performance and compared it with LBFGS, SD and CG. Several choices

of $k$ were explored. Table 4 shows results for the SRBCT data-set. In general, the bound performs best but slows down for superfluously large values of $k$. Steepest descent and conjugate gradient are slow yet obviously do not vary with $k$. Note that each iteration takes less time with smaller $k$ for the bound. However, we are reporting overall runtime which is also a function of the number of iterations. Therefore, total runtime (a function of both) may not always decrease/increase with $k$.

| $k$ | 1 | 2 | 4 | 8 | 16 | 32 | 64 |
|---|---|---|---|---|---|---|---|
| LBFGS | 1.37 | 1.32 | 1.39 | 1.35 | 1.46 | 1.40 | 1.54 |
| SD | 8.80 | 8.80 | 8.80 | 8.80 | 8.80 | 8.80 | 8.80 |
| CG | 4.39 | 4.39 | 4.39 | 4.39 | 4.39 | 4.39 | 4.39 |
| Bound | **0.56** | **0.56** | 0.67 | 0.96 | 1.34 | 2.11 | 4.57 |

Table 4: Convergence time in seconds as a function of $k$.

**Additional latent-likelihood results** For completeness, Figure 5 depicts two additional data-sets to complement Figure 2. Similarly, Table 5 shows all experimental settings explored in order to provide the summary Table 2 in the main article.

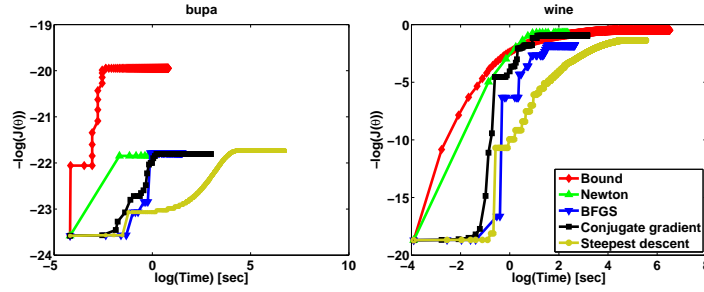

Figure 5: Convergence of test latent log-likelihood for bupa and wine data-sets.

| Data-set | ion | | | | bupa | | | | hepatitis | | | |
|---|---|---|---|---|---|---|---|---|---|---|---|---|
| Algorithm | m = 1 | m = 2 | **m = 3** | m = 4 | m = 1 | **m = 2** | m = 3 | m = 4 | m = 1 | **m = 2** | m = 3 | m = 4 |
| BFGS | -4.96 | -5.55 | -5.88 | -5.79 | -22.07 | -21.78 | -21.92 | -21.87 | -4.42 | -5.28 | -4.95 | -4.93 |
| SD | -11.80 | -9.92 | -5.56 | -8.59 | -21.76 | -21.74 | -21.73 | -21.83 | -4.93 | -5.14 | -5.01 | -5.20 |
| CG | -5.47 | -5.81 | -5.57 | -5.22 | -21.81 | -21.81 | -21.81 | -21.81 | -4.84 | -4.84 | -4.84 | -4.84 |
| Newton | -5.95 | -5.95 | -5.95 | -5.95 | -21.85 | -21.85 | -21.85 | -21.85 | -5.50 | -5.50 | -5.50 | -4.50 |
| Bound | -6.08 | -4.84 | **-4.18** | -5.17 | -21.85 | **-19.95** | -20.01 | -19.97 | -5.47 | **-4.40** | -4.75 | -4.92 |

| Data-set | wine | | | | SRBCT | | | |
|---|---|---|---|---|---|---|---|---|
| Algorithm | m = 1 | m = 2 | **m = 3** | m = 4 | m = 1 | m = 2 | m = 3 | **m = 4** |
| BFGS | -0.90 | -0.91 | -1.79 | -1.35 | -5.99 | -6.17 | -6.09 | -6.06 |
| SD | -1.61 | -1.60 | -1.37 | -1.63 | -5.61 | -5.62 | -5.62 | -5.61 |
| CG | -0.51 | -0.78 | -0.95 | -0.51 | -5.62 | -5.49 | -5.36 | -5.76 |
| Newton | -0.71 | -0.71 | -0.71 | -0.71 | -5.54 | -5.54 | -5.54 | -5.54 |
| Bound | -0.51 | -0.51 | **-0.48** | -0.51 | -5.31 | -5.31 | -4.90 | **-0.11** |

Table 5: Test latent log-likelihood at convergence for different values of $m \in \{1, 2, 3, 4\}$ on ion, bupa, hepatitis, wine and SRBCT data-sets.

